# Metric Transforms and Low Rank Representations of Kernels for Fast Attention

**Timothy Chu**
Independent Researcher
timothyzchu@gmail.com

**Josh Alman**
Columbia University
josh@cs.columbia.edu

**Gary Miller**
Carnegie Mellon University
glmiller@cs.cmu.edu

**Shyam Narayanan**
Citadel Securities
shyam.s.narayanan@gmail.com

**Mark Sellke**
Harvard University
msellke@fas.harvard.edu

**Zhao Song**
Simons Institute for the Theory of Computing, UC Berkeley
magic.linuxkde@gmail.com

## Abstract

We introduce a new linear-algebraic tool based on group representation theory, and use it to address three key problems in machine learning.

1. Past researchers have proposed fast attention algorithms for LLMs by approximating or replace softmax attention with other functions, such as low-degree polynomials. The key property of these functions is that, when applied entry-wise to the matrix $QK^\top$, the result is a low rank matrix when $Q$ and $K$ are $n \times d$ matrices and $n \gg d$.

   This suggests a natural question: what are all functions $f$ with this property? If other $f$ exist and are quickly computable, they can be used in place of softmax for fast subquadratic attention algorithms. It was previously known that low-degree polynomials have this property. We prove that low-degree polynomials are the *only* piecewise continuous functions with this property. This suggests that the low-rank fast attention only works for functions approximable by polynomials. Our work gives a converse to the polynomial method in algorithm design.

2. We prove the first full classification of all positive definite kernels that are functions of Manhattan or $\ell_1$ distance. Our work generalizes, and also gives a new proof for, an existing theorem at the heart of kernel methods in machine learning: the classification of all positive definite kernels that are functions of Euclidean distance.

3. The key problem in metric transforms, a mathematical theory used in geometry and machine learning, asks what functions transform pairwise distances in metric space $M$ to metric space $N$ for specified $M$ and $N$. We prove the first full classification of functions that transform Manhattan distances to Manhattan distances. Our work generalizes the foundational work of Schoenberg, which fully classifies functions that transform Euclidean to Euclidean distances.

We additionally prove results about stable-rank preserving functions that are potentially useful in algorithmic design, and more. Our core tool for all our results is a new technique called the representation theory of the hyperrectangle.

# 1   Introduction

Kernel methods in linear algebra have numerous applications throughout computer science and machine learning. Consider the following basic questions in this area.

(A) Given any set of low-dimensional points $x_1, x_2, \cdots, x_n, y_1, y_2, \ldots, y_n \in \mathbb{R}^d$ and a function $f : \mathbb{R} \to \mathbb{R}$, is there a small $k < n$ and a function $F : \mathbb{R}^d \to \mathbb{R}^k$ such that $f(\langle x_i, y_j \rangle) = \langle F(x_i), F(y_j) \rangle$ for all $i$ and $j$? Equivalently, when $f$ is applied entry-wise to a low-rank (kernel) matrix, is the result always low-rank? What about *approximately* low-rank?

(B) Given any set of points $x_1, x_2, \cdots, x_n$ and a (kernel) function $f : \mathbb{R} \to \mathbb{R}$, is there some function $F$ such that $f(\|x_i - x_j\|_1) = \langle F(x_i), F(x_j) \rangle$ for all $i$ and $j$? Equivalently, is $f$ a positive definite Manhattan kernel?

(C) Given any set of points $x_1, \cdots x_n$, from semi-metric spaces $\mathcal{X}$ and $\mathcal{Y}$, for which functions $f$ does there exist a function $F$ such that $f(\text{dist}_{\mathcal{X}}(x_i, x_j)) = \text{dist}_{\mathcal{Y}}(F(x_i), F(x_j))$ for all $i$ and $j$? Equivalently, which functions $f$ give a metric transform [DL09] between $\mathcal{X}$ and $\mathcal{Y}$?

Question (A) relates to the study and effectiveness of polynomial kernels in machine learning. These kernels have many applications [Sou10], for instance in speeding up attention in LLMs [AS23, KMZ23, TBY$^+$19, KVPF20, AS24b, AS24c, SSWZ23a], in NLP for improving the quality of learning algorithms [KM03, GE08, CHC$^+$10] and basic computations [VSP$^+$17] during training. Oftentimes, if $f$ is not a polynomial, one may even approximate it by a polynomial (for instance by truncating its Taylor expansion) or use sketching [HAS20, SWYZ21, SZZ24, SYZ24] in order to achieve some of the benefits of polynomial kernels in exchange for worse accuracy guarantees. This has been particularly effective for the neural tangent kernel [JGH18], Gaussian kernel [NJW02, RR08, AKK$^+$20, HSW$^+$22], generalized T-Student kernel [BTF04], Cauchy kernel [RR08], and power kernel [FS03].

One can verify that if $f$ is a polynomial, then one can achieve $k \leq d^{\deg(f)}$ in question (A). Beyond just kernel methods, this fact has been used to design efficient algorithms throughout computer science using a technique called 'the polynomial method in algorithm design' (see e.g., [AWY14, CW16, ACW16, Wil18, Alm19, ACSS20, AS23, AS24c, AS24b], and Section 2 below for more examples). Determining which other functions $f$ have this key property can help to extend these phenomena to more settings.

Kernel methods, and linear-algebraic computations related to the kernel matrix such as those involved in Question (B), are very popular in modern machine learning. In some applications, such as spectral clustering [vL07, NJW02], semi-supervised learning [Zhu05a, Zhu05b, LSZ$^+$19, SSLL23, SSL24], Laplacian system solving in geometric graphs [ACSS20] and kernel support vector machines [GSZ23], one needs to explicitly compute the kernel matrix and corresponding function $F$. On the other hand, for many other applications such as regression or classification algorithms, it suffices to implicitly maintain the kernel matrix, or simply prove that the function $F$ exists [Smo96, SSB$^+$97]; several prominent recent examples are neural tangent kernel regression [BPSW21, SYZ21, MOSW22, ALS$^+$23, GLS$^+$24, LLSS24], tensor kernel regression [Zha22, RSZ22, SZZ24], polynomial kernels [SWYZ21, HAS20, AKK$^+$20, SYZ24], and signal interpolation [CKPS16, SSWZ23b]. This motivates question (B), where we ask whether $F$ exists, but not how efficient it is.

The metric transforms referenced in Question (C) arise naturally in many settings where one wants to transform a set of points from a metric space while maintaining some of the metric structure between them. The field was pioneered by Schoenberg [Sch38, Sch42, Sch35] and Von Neumann [NS41]. Very broadly, this allows one to take advantage of algorithmic tools in both metric spaces simultaneously [DL09]. This approach has proven useful in many areas including visualizing the geometry of BERT [RYW$^+$19], computer vision [FLH15, KZR16, KCC17, WZF05], clustering [MMR19], sketching and embedding norms [AKR15, IMS17], terminal embedding [MMMR18, NN19, CN21], low dimensional embeddings via JL transform [AC06, DKS10], mean estimation [LNRW19] nearest neighbor search [AIR18], generative models [XZZ18], data-sensitive distances in clustering [CMS20], neural networks [Orr96], harmonic analysis [Aro50, LLLH18, KW71], kernel methods [SSB$^+$97], distance oracle [DSWZ22], and PDE theory [FS98, CFW12].

Many researchers consider metric transforms when the input and output space are Euclidean, since more is known about metric transforms in this setting. However, metric transforms between other

metrics could have equally rich algorithmic applications, and question (C) generalizes this beyond just Euclidean metrics. We could think of distance metrics in two semi-metric spaces $\mathcal{X}$ and $\mathcal{Y}$, and the function pair $(f, F)$ can be viewed as a transform from $\mathcal{X}$ to $\mathcal{Y}$.

In all these settings, there is a gap in our current understanding of what kernel functions can be used. For instance, there are a number of functions where it is *not known* whether they are positive definite Manhattan kernels which could be used in classification, semi-supervised learning, and other similar tasks. We will see that a common suite of mathematical tools can be used to address all these different gaps. Most of our results prove that our understanding is complete, and that, for instance, the functions we know to be positive definite Manhattan kernels are, in fact, the only ones. This finally completes our understanding of these important classes of functions. That said, in the setting of preserving the stable rank of matrices, we will find that there are functions that are not polynomials, but that surprisingly do preserve the stable ranks of important matrices. Before we get into our technique in more detail, we first describe our main results in context.

**Roadmap.** In Section 2, we prove that the only functions which always yield a low-rank matrix when applied entry-wise to a low-rank matrix are low-degree polynomials, and explain the application to transformers. In Section 3, we give a classification of positive definite kernels with Manhattan distance input. In Section 4, we categorize all functions which transform Manhattan distances to Manhattan distances or squared Euclidean distances. In Section 5, we briefly introduce the core tool of this work. In Section 6, we give a conclusion of our work. In Section 7, we discuss the limitations of our work. In Section 8, we discuss the societal impacts of our work.

## 2 Fast Attention and the Polynomial Method

Fast attention computations in transformers and LLMs [AS23, AS24c, AS24b, AS24a, LSS+24, HWL+24, LSSZ24a, LSSZ24b, KVPF20, TBY+19] use the *polynomial method* as a key ingredient. This is a powerful technique for designing algorithms and constructing combinatorial objects. It states that applying a low-degree polynomial entry-wise to a low rank matrix yields another low-rank matrix. Examples of these low rank matrices include $QK^\top/\sqrt{d}$ for $n \times d$ matrices $Q$ and $K$ with $n \gg d$, which is a key matrix for attention computations in transformers and LLMs.

Fast attention computations rely on a polynomial approximation to the exponential function [AS23] combined with the polynomial method. Past researchers [KMZ23, SSWZ23a, TBY+19, KVPF20] suggested replacing the exponential function in softmax attention with a general kernel function $f$. When $f$ is a low-degree polynomial, researchers leveraged the polynomial method to create fast algorithms for polynomial attention in LLM computations [KMZ23, TBY+19, KVPF20]. The work of [KMZ23] showed experimentally that polynomial attention has faster training and inference times, with little loss in quality on large language models.

**Fact 2.1** (The polynomial method, folklore; see e.g. [CLP17]). *Suppose $f : \mathbb{R} \to \mathbb{R}$ is a polynomial of degree $d$. Then, for any matrix $M \in \mathbb{R}^{n \times n}$ of rank $r$, letting*

$$k := 2\binom{r + \lfloor d/2 \rfloor - 1}{\lfloor d/2 \rfloor},$$

*the matrix $M^f \in \mathbb{R}^{n \times n}$ given by $M_{i,j}^f := f(M_{i,j})$ has $\mathrm{rank}(M^f) \leq k$.*

*For instance, if $r = \log_2 n$ and $d < o(\log_2 n)$, then*

$$\mathrm{rank}(M^f) < n.$$

The definition of the polynomial method inspires the following definition:

**Definition 2.2** (Preserve low-rank matrices). *For a function $f : \mathbb{R} \to \mathbb{R}$ and positive integer $n$, we say $f$ preserves low-rank $n \times n$ matrices if, for every matrix $M \in \mathbb{R}^{n \times n}$ with $\mathrm{rank}(M) \leq \lceil \log_2(n) \rceil + 1$, the entry-wise application $M^f \in \mathbb{R}^{n \times n}$ given by $M_{i,j}^f := f(M_{i,j})$ has*

$$\mathrm{rank}(M^f) < n.$$

It follows from the polynomial method that low-degree polynomials preserve low rank. Fast attention computations [AS23] rely on this low rank preservation property. For any function $f$ that preserves

low rank, if the low rank decomposition of $M^f$ can be efficiently computed, one can create a replacement for attention that runs in almost linear time in the sequence length $n$. This is a significant improvement over the quadratic time algorithms necessary for (unbounded) softmax attention in LLM models (implicit in [KMZ23, AS23, TBY$^+$19, SSWZ23a]).

This motivates the question:

**Question 2.3.** *Is it possible to generalize the polynomial method (Fact 2.1) to functions $f$ other than polynomials?*

In other words, are there functions $f$ which are not polynomials, but such that if one starts with any low-rank matrix $M$, and applies it entry-wise yielding the matrix $M^f$, then $M^f$ also has low rank? If such a function existed, it could allow for faster transformers with a wider variety of attention functions, along with many other algorithmic applications.

We prove that low-degree polynomials are the only piecewise continuous functions $f$ that preserve low rank. This suggests that the low-rank approach to fast attention calculations [AS23, AS24b, AS24c] and fast polynomial attention algorithms [KMZ23] can *only* work for functions that are approximations of polynomials.

The polynomial method can also be used in algorithm design to design the fastest known algorithms for a variety of different, important problems, including: batch Hamming Nearest Neighbor Search [ACW16], the Orthogonal Vectors problem from fine-grained complexity [AWY14, CW16], All-Pairs Shortest Paths [Wil18, CW16], the lightbulb problem in which one wants to find a planted pair of correlated vectors among a collection of random vectors [Val12, KKK18, Alm19], computational problems related to kernel methods in spectral clustering and semi-supervised learning [ACSS20], and some stable matching problems [MPS16]. In all these works, one starts with a matrix $M$ describing the input data which has low rank, and one transforms it into a matrix like $M^f$ which 'amplifies' the key properties of the data while still having low rank. A similar approach has also been used in the theory of polynomial kernels, such as in algorithms for transformers in NLP [VSP$^+$17, DCLT18, RNS$^+$18, RWC$^+$19, BMR$^+$20, CND$^+$22, ZRG$^+$22, AS23, AS24c, KMZ23, HJK$^+$23, HSK$^+$24, HLSL24, HWL$^+$24], and to bound the ranks of matrices which arise in other settings, such as in the recent resolution of the Cap Set Conjecture from extremal combinatorics [CLP17, EG17], and in recent proofs that Hadamard and Fourier transforms have low 'matrix rigidity' [AW17, DE19, DL19].

For a function $f$ to be effective in the polynomial method as described above, it is necessary (but usually not sufficient) that $f$ preserves low-rank $n \times n$ matrices in the sense of Definition 2.2. Indeed, in all the aforementioned applications of the polynomial method, such as the algorithm of [ACW16] and the application to transformers that we described above, the original matrix $M$ describing the data can have rank greater than $\log_2 n$. The details of how low the rank of $M^f$ must be can vary in the different applications, but it is always necessary that $M^f$ has less than *full* rank (i.e., $\mathrm{rank}(M^f) < n$).

## 2.1 Converse for the Polynomial Method

Our starting point is the recent work in mathematics by Guillot, Khare, and Rajaratnam [GKR17], which partially answers Question 2.3 negatively. This shows that Fact 2.1 cannot be generalized in many settings.

**Theorem 2.4** ([GKR17, Theorem B], Informal). *Recall that in Fact 2.1, $k$ is the target rank of the matrix $M^f$. Fact 2.1 is tight, i.e. its converse is true, when either $f$ is $(n-1)$-times differentiable and $k < n-1$, or if $M^f$ is required to be positive semi-definite and $k < n-3$.*

This past work has gaps where $f$ might still result in matrices without full rank, especially since the requirement that $f$ is $(n-1)$-times differentiable is quite restrictive. Common functions in machine learning like ELU [CUH15, KVPF20, CLD$^+$20], SELU [KUMH17, ZLZ24], and ReLU [HSM$^+$00, LSS$^+$20, ZZP$^+$21, ZLZ24] are not second-differentiable everywhere, so Theorem 2.4 doesn't apply to them. One might imagine getting around this differentiability restriction by using the second part of Theorem 2.4, but unfortunately the matrices $M^f$ involved in fast attention computations are not required to be positive semi-definite. So this second part of the theorem does not apply to fast attention, which is a core application of functions that preserve low rank.

Our first result plugs these gaps, showing that Fact 2.1 cannot be generalized in the settings left open by [GKR17]. (See Section C.8 below where we state [GKR17, Theorem B] more formally and compare it with our Theorem 2.5 in more detail.)

**Theorem 2.5** (Informal statement of Theorem C.11). *Suppose the function $f : \mathbb{R} \to \mathbb{R}$ does not have any essential discontinuities of the first kind[1]. If $f$ preserves low-rank $n \times n$ matrices, then $f$ is a polynomial of degree at most $\lceil \log_2(n) \rceil$.*

This shows that functions $f$ without essential discontinuities of the first kind, which are also not polynomials, do not preserve low-rank $n \times n$ matrices, and only polynomials of degree less than $\lceil \log_2(n) \rceil$ can preserve low-rank $n \times n$ matrices. The class of functions without these essential discontinuities of the first kind is very rich, and includes all *piecewise continuous* functions; it is hard to imagine a reasonable kernel function which is not piecewise continuous. Hence, one cannot hope to improve on the polynomial method by extending it to other functions without essential discontinuities of the first kind.

We conjecture that Theorem 2.5 holds for *all* functions $f : \mathbb{R} \to \mathbb{R}$ (i.e., that if $f$ has an essential discontinuity of the first kind, then it also does not preserve low-rank matrices). Functions $f$ *with* an essential discontinuity of the first kind are not interesting in our setting since they cannot be efficiently evaluated.

We note that there is a small constant-factor gap between the degree which Fact 2.1 tells us is sufficient for a polynomial to preserve low-rank $n \times n$ matrices, and the degree which Theorem 2.5 says is necessary: for instance, Fact 2.1 says that polynomials of degree at most $\frac{1}{2} \log_2(n)$ suffice, since

$$\binom{\frac{5}{4} \log_2(n)}{\frac{1}{4} \log_2(n)} \ll n,$$

whereas Theorem 2.5 says that degree less than $\log_2(n)$ is necessary. We leave open the question of closing this gap, although we note that the constant factor in front of the polynomial degree does not play a major role in most of the aforementioned applications of Fact 2.1.[2]

## 2.2 Weaker Polynomial Methods

Fact 2.1 being essentially tight rules out one way to try to generalize the polynomial method. It is natural to ask whether we can get around this by weakening our constraint on the function $f$. There are many properties of matrices which can be taken advantage of in the design of fast algorithms, and if we can show that $M^f$ has any of these properties, it could still lead to improvements in the aforementioned applications.

**Approximate Low Rank** We first study functions $f$ which, when applied entry-wise to a low-rank matrix $M$, always result in an *approximately* low-rank matrix $M^f$. As we mentioned earlier, approximating a non-polynomial kernel function $f$ by a polynomial is a common technique for taking advantage of the properties of polynomial kernels; when $f$ can be well-approximated by a polynomial, then $M^f$ has approximately low rank for this reason. This raises the question: can functions $f$ which cannot be well-approximated by a polynomial also result in approximately low-rank matrices?

Our next result answers this question in the negative: the only functions which approximately preserve low rank are approximate polynomials. In other words, if $f$ is not approximately a polynomial, then such an algorithmic approach cannot succeed, as $f$ applied entry-wise to a matrix is not close to low rank.

We say a matrix $M$ is approximately low-rank if the ratio of its smallest and largest eigenvalues is small; if $M$ were not full rank, then this ratio would be 0. If $M$ is approximately low-rank in this sense, then fast algorithms for manipulating it follow by using low-rank approximation or approximate

subspace finding algorithms to find low-rank approximations for the matrix. Analogously, we say $f$ is approximately a polynomial if its finite differences are small[3]; recall that the $d$-th order finite differences are 0 for any polynomial of degree $< d - 1$.

**Theorem 2.6** (Main result, informal statement of Theorem D.3). *Let $d = \lceil \log_2 n \rceil$, and let $\delta \in (0, 1)$ be sufficiently small. Suppose $f : \mathbb{R} \to \mathbb{R}$ is a real analytic function which $\delta$-approximately preserves low-rank matrices, i.e., $\frac{\min_{i \in [d]} |\lambda_i(M^f)|}{\max_{i \in [d]} |\lambda_i(M^f)|} \leq \delta/n$ for all rank $d + 1$ matrices $M$.*

*Then, the $d^{th}$ order finite difference of $f$, evaluated at $a \in \mathbb{R}$, for sufficiently small gaps, is bounded above by $\delta \cdot K_a$. Here, $K_a > 0$ is a scaling factor with the property that if $f$ is rescaled by a factor of $c > 0$ then $K_a$ is also rescaled by $c$.*

A dependence on a scaling factor $K_a$ is necessary since, if $f$ is rescaled by $c$, this rescales the finite differences of $f$ by $c$, but does not change the ratio of any two eigenvalues of any matrix $M^f$. In Theorem D.3 we also prove a similar result if $f$ is Lipschitz (and not necessarily real analytic).

**Stable Rank**  Our first two results, Theorem 2.5 and Theorem 2.6, both *ruled out* approaches to generalizing the polynomial method. Finally, we find one important property of matrices for which we *can* strictly generalize the polynomial method: stable rank.

**Definition 2.7.** *For a matrix $M \in \mathbb{R}^{n \times n}$, its stable rank is defined as $\mathrm{srank}(M) := \frac{\|M\|_F^2}{\|M\|_2^2}$, where $\|M\|_F$ denotes the Frobenius norm of matrix $M$ and $\|M\|_2$ denotes the spectral norm of matrix $M$.*

It is known that $\mathrm{srank}(M) \leq \mathrm{rank}(M)$, but there are example matrices where $\mathrm{srank}(M) \ll \mathrm{rank}(M)$. Moreover, matrices with low stable rank can be manipulated quickly in many applications; for instance, low stable rank matrices are a useful tool in data mining and the study of Banach spaces [MSS17], and very efficient sketching methods are known for matrices with small stable rank [CNW15].

**Theorem 2.8** (Informal statement of Theorem J.2). *Let $M \in \mathbb{R}_{>0}^{n \times n}$ be a matrix, and suppose $f : \mathbb{R}_{>0} \to \mathbb{R}_{>0}$ has the property that for any entry $z$ of $M$, $\frac{1}{\sqrt{L}} \cdot z \leq f(z) \leq \sqrt{L} \cdot z$ for some $L \in \mathbb{R}_{>0}$. Then, $\mathrm{srank}(M^f) \leq L^2 \cdot \mathrm{srank}(M)$.*

Consider, for instance, the matrices which arise in polynomial method applications [ACW16]; these are matrices $M \in \mathbb{R}^{n \times n}$ where each entry is in the interval $[1, O(\log n)]$. For these matrices, functions like $f(x) = x^c$, for any constant $c > 0$, which are *not* a polynomial when $c$ is not an integer, still preserve stable rank (they satisfy the condition of Theorem 2.8 with $L = \mathrm{poly} \log(n)$). By contrast, such bounds on the entries of $M$ do not impact our earlier results, and so such functions do not preserve rank or approximately preserve rank for these matrices.

Unfortunately, it is not clear how to apply this to speed up the polynomial method applications we discussed earlier. Most known applications of stable rank require one to have access to the entire matrix $M^f$ (in order to, for instance, apply sketching), whereas we are aiming for algorithms whose running time is sublinear in the number of entries of $M^f$. Nonetheless, this is an exciting avenue where one can strictly generalize the polynomial method, and we believe it will have interesting algorithmic applications, and further motivates algorithmic applications of stable rank.

Theorem 2.8 tells us that functions which do not grow too quickly preserve stable rank, although the desired rate of growth depends on the matrix entries. We also prove a complementary result about functions which do grow very quickly: In Theorem J.5 we prove that any super-polynomial function which grows like $x^{\log^c(x)}$ for any $c > 0$ does not preserve low stable rank, and applying it entry-wise to an $n \times n$ matrix of rank $O(\log n)$ can result in a matrix of stable rank $> n - 1$. Hence, there is a limit to how much one could improve our Theorem 2.8.

## 3   Kernel Methods

Our second main application of our techniques is to the study of kernel methods in machine learning. Much of the prior work on kernels methods focuses in the Euclidean distance setting. Our new result

shows how to classify kernels in the Manhattan distance setting. We start with defining positive definite kernel.

**Definition 3.1** (Positive definite Manhattan/Euclidean kernel). *A function $f$ is a positive definite $\ell_p$ kernel if, for any $x_1, \ldots x_n \in \mathbb{R}^d$ for any $n$ and $d$, the matrix $M \in \mathbb{R}^{n \times n}$ with*

$$M_{i,j} = f(\|x_i - x_j\|_p)$$

*is positive semi-definite.*

*For $p = 1$, we also say $f$ is a positive definite Manhattan kernel, and for $p = 2$ we call it a positive definite Euclidean kernel.*

Equivalently, $f$ is a positive definite $\ell_p$ kernel if and only if there exists a function $F : \mathbb{R}^d \to \mathcal{H}$ such that: $\langle F(x), F(y) \rangle = f(\|x - y\|_p)$ for all $x, y \in \mathbb{R}^d$ for all $d$. Note that $\mathcal{H}$ represents Hilbert space. The proof of the equivalence can be found in [Sch42]. Positive definite kernels are used in machine learning to separate data embedded in $\mathbb{R}^d$ using linear separator techniques, when the initial data is not linearly separable [Smo96, SSB$^+$97, SOW01, SS01]. In other words, a positive definite kernel can map points in $\mathbb{R}^d$ which are not linearly separable, to points in potentially higher dimensions which are linearly separable. Finding such an embedding is not an easy task in general, but kernel methods solve this problem.

Many regression algorithms *require* the kernel to be positive definite [Cut09, HTF09]. The key idea is to pick a function $f$ based on the application so that a function $F$ like the one in Definition 3.1 can be found which maps the data points to vectors of possible higher dimensions, after which linear separation can be performed efficiently on these higher dimensional points.

Interestingly, linear separator algorithms such as the widely used *Support Vector Machines* (SVMs) [CV95] can separate the data efficiently as long as $\langle F(x), F(y) \rangle$ is easily computed for any $x, y \in \mathbb{R}^d$, even if $F$ itself cannot be easily computed. By definition of the positive-definite kernel $f$, we know that

$$\langle F(x), F(y) \rangle = f(\|x - y\|_p),$$

which allows us to compute $\langle F(x), F(y) \rangle$ quickly by instead computing $f(\|x - y\|_p)$. In other words, in order to apply linear separator algorithms, it suffices to know that an $F$ *exists*, and not necessarily know what it is or how to compute it.

The main known result behind kernel methods is a full classification of all positive-definite Euclidean kernels in terms of completely monotone functions, which are defined as follows:

**Definition 3.2** (Completely monotone functions [Ber29]). *A function $f : \mathbb{R}_{\geq 0} \to \mathbb{R}_{\geq 0}$ is completely monotone if*

$$(-1)^k f^{(k)}(x) \geq 0$$

*for all $k \geq 0, x > 0$, and $f(0) \geq \lim_{x \to 0^+} f(x)$.*

An example of a completely monotone function is $f(x) = e^{-x}$. Prior work [Mer09, Sch42, Sch38, SSB$^+$97, SSM98, SOW01] shows that function $f : \mathbb{R} \to \mathbb{R}$ is a positive-definite Euclidean kernel (Definition 3.1) if and only if $f(\sqrt{x})$ is a completely monotone function (Definition 3.2). A natural question to ask is

**Question 3.3.** *Is there a result that classifies all positive definite Manhattan kernels?*

In our paper, we classify all positive-definite Manhattan kernels. These kernels are widely used in machine learning for physical and chemical applications [FLLA15, Lil18, LRRK15]. A notable example of such a kernel is the Laplace kernel $f_\sigma(x) = e^{-\sigma x}$ which is commonly used in classification tasks [BMM18]. However, a full description of all positive-definite Manhattan kernels was not known before our work. In this work, we answer Question 3.3 positively:

**Theorem 3.4** (Main result, informal statement of Theorem G.2). *$f$ is a positive definite Manhattan kernel (Definition 3.1) if only if $f(x)$ is completely monotone (Definition 3.2).*

Theorem 3.4 classifies all positive-definite kernels when the input distance is Manhattan. It was previously known that completely monotone functions are positive definite Manhattan kernels [Sch38,

Ass80, DL09], but it was not known these were the only such functions. Interestingly, our new classification is similar to the classification result for Euclidean kernels, but without a square root applied to the input. Prior to our result, one could have imagined that there are other positive definite Manhattan kernels to use in SVMs than were previously known. However, our result shows that there are no other such kernels.

We note that our proof techniques also give a new proof of the known result classifying all positive definite Euclidean kernels. This known result is a core insight at the heart of kernel methods in machine learning [SSB+97, SSM98], but traditional proofs tend to use methods related to infinite dimensional harmonic analysis [BCR84].

## 4  Metric Transforms

Our final application of our techniques is to *metric transforms*, a mathematical notion introduced by Von Neumann and Schoenberg [NS41].

**Definition 4.1** (Metric transform)**.** *Suppose $\mathcal{X}$ and $\mathcal{Y}$ are semi-metric spaces[4]. Function $f$ **transforms** $\mathcal{X}$ to $\mathcal{Y}$ if, for any finite set $S \subseteq \mathcal{X}$, there is a function $F : \mathcal{X} \to \mathcal{Y}$ such that $f(d_{\mathcal{X}}(x_1, x_2)) = d_{\mathcal{Y}}(F(x_1), F(x_2))$, for all $x_1, x_2 \in S$.*

As we discussed earlier, metric transforms arise naturally in many settings where one wants to transform a set of points from a metric space while maintaining some of the metric structure between them, and they have proven useful for algorithm design in many areas.

Typically we have particular metric spaces $\mathcal{X}$ and $\mathcal{Y}$ of interest, and would like to determine which functions transform $\mathcal{X}$ to $\mathcal{Y}$. This leads to the key question in metric transforms:

**Question 4.2.** *For a given semi-metric space $\mathcal{X}$ and a given semi-metric space $\mathcal{Y}$, what is the full classification of functions $f$ that transform $\mathcal{X}$ to $\mathcal{Y}$?*

Metric transforms in the special case where $\mathcal{X}$ and $\mathcal{Y}$ are both Euclidean distances[5] or close variants are well-studied. Related to Schoenberg and Von Neumann's work [NS41], Schoenberg [Sch38] classified all functions that transform Euclidean distances to Euclidean distances. One natural question arises: what is the theory of metric transforms for non-Euclidean metrics?

In the case when $\mathcal{X}$ is Manhattan (or $\ell_1$) distance, and $\mathcal{Y}$ is Euclidean distance, Schoenberg [Sch38] provided a partial categorization of functions that transform Manhattan distance to Euclidean distance. This was followed by Assouad's work in 1980, which provided a partial categorization of functions that transform Manhattan distances to Manhattan distances [Ass80]. This setting is particularly well-motivated in physical applications. For instance, recent work [GSDV17] studied the problem of inferring a force vector given a collection of example configurations via kernel ridge regression; in order to encode certain desired symmetries ('axis reflections') in the problem, Manhattan preserving functions must be used to define the kernel. Our work on metric transforms completes the partial categorizations of Schoenberg and Assouad, and proves their partial categorization is a full categorization.

Our main result about metric transforms is a complete classification of functions that transform Manhattan distances to Manhattan distances. First, we need to define Bernstein functions:

**Definition 4.3** (Bernstein functions [Ber29])**.** *A function $f : \mathbb{R}_{\geq 0} \to \mathbb{R}_{\geq 0}$ is Bernstein if $f(0) = 0$ and its derivative $f'$ is completely monotone (see Definition 3.2) when restricted to $\mathbb{R}^+$. Equivalently, a function $f$ is Bernstein if:*

- *1. $(-1)^k \frac{\mathrm{d}^k f(x)}{\mathrm{d}x^k} \leq 0$ for all $k \geq 1, x \geq 0$;*

- *2. $f(x) \geq 0$ for all $x \geq 0$; and*

- *3.* $f(0) = 0.$[6]

Now we are ready to state our main result:

**Theorem 4.4** (Main result, classifying all Manhattan metric transforms, informal version and combination of Theorem E.2 and F.3). *For a function $f : \mathbb{R}_{\geq 0} \to \mathbb{R}_{\geq 0}$, the following are equivalent:*

1. *$f$ is Bernstein.*
2. *$f$ transforms Manhattan distances to Manhattan distances.*
3. *$f$ transforms Manhattan distances to squared Euclidean distances.*

It was previously known that Bernstein functions transform Manhattan distances to Manhattan distances [Ass80], and that they transform Manhattan distances to squared Euclidean distances [Sch38], but in both cases, it was not previously known that these were the only such functions. It was previously conceivable that, in situations where one needs a metric transform involving Manhattan spaces, but Bernstein functions do not suffice, one could find other suitable metric transforms; our Theorem 4.4 rules out such a possibility. This also has a number of simple consequences, for instance: given any $n$ points $x_1, \ldots x_n$ in the metric space $(\mathbb{R}^d, \ell_1)$ for any $d$, one can use our construction in Theorem 4.4 to explicitly calculate a finite dimensional embedding $F : \mathbb{R}^d \to \mathbb{R}^{2^d}$ such that $\|F(x_i) - F(x_j)\|_1 = f(\|x_i - x_j\|_1)$.

# 5 Core Tool: Representation Theory of the Real Hyperrectangle

Our core mathematical tool to tackle all three problems is a new technique we call the *representation theory of the hyperrectangle*. Given a $d$ dimensional hyperrectangle (which is just a high dimensional rectangle), consider the matrix $D$ where the $ij^{th}$ entry of the matrix is the Manhattan distance between the $i^{th}$ and $j^{th}$ vertex of the hyperrectangle. We prove this matrix has three key properties:

1. It is a $2^d \times 2^d$ matrix whose rank is $d + 1$, and thus it is a low rank matrix.

2. This matrix is filled with Manhattan distances between points.

3. Applying $f$ entry-wise to this matrix does not change the eigenvectors of this matrix, which are always the columns of the so-called Hadamard matrices [HW78].

We note that the last property is particularly useful for us: it allows us to provide a closed formula for the eigenvectors and eigenvalues of $D^f$. This is particularly useful because all of our key questions (on low rank preservation, kernels, and metric transforms) can be viewed as questions about the eigenvalues of certain matrices after a function is applied entrywise.

The last property can be verified by linear algebra computation, but it can also be seen as a consequence of group representation theory. Thus, we call our approach the representation theory of the hyperrectangle. For proofs of all three properties, refer to Appendix B.1 and I.

# 6 Conclusion

We demonstrate that low-degree polynomials are the only functions that consistently result in a low-rank matrix when applied entry-wise to an existing low-rank matrix, and discuss applications to transformers and LLMs. Additionally, we classify all positive definite kernels that utilize Manhattan distance as their input, enhancing the theoretical framework for applications in various machine learning tasks. Furthermore, we provide a complete categorization of functions capable of transforming Manhattan distances into either Manhattan distances or Euclidean distances. We do all three tasks using a new linear algebraic tool called the representation theory of the hyperrectangle. Our findings not only advance the theoretical understanding of attention and kernel methods, but also open up new possibilities for their application in fields such as computational biology and algorithm design. This work completes the theoretical landscape of Manhattan to Manhattan metric transforms, and utilizes a sophisticated blend of mathematical techniques from several domains.

## 7 Limitations

In this paper, we identify a few areas that require further exploration. Firstly, there remains a small constant-factor gap between Fact 2.1 and Theorem 2.5 that has not been fully explored; details can be found in the last paragraph of Section 2.1. Additionally, the application of the stable rank results presents an ongoing challenge, as discussed in the second-last paragraph of Section 2.2. Finally, our analysis primarily focuses on LLMs, kernel methods, and metric transforms, potentially limiting its applicability to other methodologies.

## 8 Societal Impacts

This paper contributes positively by providing a deeper and more comprehensive study of kernel functions and completes the theory of Manhattan to Manhattan metric transforms, a problem that has persisted since 1980 due to Assouad's work. It opens up numerous algorithmic applications, potentially including large language models (LLMs), and offers a new direction for designing faster algorithms using stable rank results. However, the practical application of these results remains an open area, requiring additional time and effort to fully realize their potential.

## 9 Acknowledgments

The authors would like to thank Amol Aggarwal, Ainesh Bakshi, Robi Bhattacharjee, Jerry Yao-Chieh Hu, Rajesh Jayaram, Monique Laurent, Roie Levin, Yingyu Liang, Han Liu, Ryan O'Donnell, Zhenmei Shi, Kevin Tian, Alex Wang, Yu Wang, Yufa Zhou, Lichen Zhang, Di Zhu, and Goran Zuzic for support and helpful comments. The authors would like to thank Yufa Zhou for writing the conclusion and checklist of the paper. This work was mostly done when Timothy Chu was at Carnegie Mellon University, and later at Google, and when Shyam Narayanan was a student at MIT. Josh Alman is supported in part by NSF Grant CCF-2238221 and a Google Research Scholar Award. Gary Miller is supported in part by NSF Grant CCF-1637523. Shyam Narayanan is supported by an NSF Graduate Fellowship and a Google Fellowship. For more information related to the paper and adjacent topics, see https://www.youtube.com/@zhaosong2031 and https://space.bilibili.com/3546587376650961.

## Footnotes

[1]Recall that $f : \mathbb{R} \to \mathbb{R}$ has an essential discontinuity of the first kind at a point $c \in \mathbb{R}$ if neither of the limits $\lim_{x \to c^+} f(x)$, nor $\lim_{x \to c^-} f(x)$, converges. By contrast, if exactly one of the two limits converges, it is called an essential discontinuity of the second kind. If both limits converge, but they don't both converge to $f(c)$, then it may be a removable discontinuity or a jump discontinuity.

[2]For instance, our running example algorithm of [ACW16] only uses an asymptotic bound on how the degree grows with the dimension of the input points, and the constant factor in front of the polynomial degree is ultimately subsumed by a '$O$' in the running time.

[3]The $d$-th order finite difference is the discrete analog of the $d$-th order derivative. For a formal definition of finite differences, see the paragraph on finite differences in Section A.2.

[4]A semi-metric satisfies all the axioms for a metric except possibly the triangle inequality; the square of the Euclidean distance gives rise to a semi-metric.

[5]When we refer to Euclidean or Manhattan distance in the remainder of this section, we always refer to distances in infinite dimensional Euclidean metric space and infinite dimensional Manhattan metric spaces, respectively.

[6]We remark that the special attention on $f(0)$ in the definitions above is a bit non-standard but are convenient for our purposes.

[7]We note that Schoenberg's criteria has a beautiful proof, which one can find one direction of in [Par12].

[8]The matrices $Q, K$ and $V$ correspond to the query, key, and value matrices, respectively, when training transformers in NLP applications. For more background, we refer the reader to [VSP+17, DCLT18, RNS+18, RWC+19, BMR+20, KKL19, CLD+20, FZS21, WLK+20, CDW+21, ZSZ+23, LWD+23]. The $n \gg d$ is a reasonable assumption in long sequence model problems, since $n$ is the length of documents and $d$ is size of each word embedding.

[9]We first construct the matrices $X \in \mathbb{R}^{n \times 2d}$ and $Y \in \mathbb{R}^{2d \times n}$ such that $M = X \times Y$. We can then compute the product $X \times Y$ in $\widetilde{O}(n^2)$ time using fast rectangular matrix multiplication [Cop82, Wil18] as long as $d < n^{0.1}$. The reason of obtaining $2d$ (instead of $d$) is due to the construction of [ACW16].

[10]They pick rank $\approx n^{0.1}$ in order to apply fast rectangular matrix multiplication as in footnote 9, although different applications of the polynomial method have aimed for different target ranks.

[11]Theorem 3 of [SOW01] is a modern restatement of Schoenberg's work in [Sch42].

## References

[AC06] Nir Ailon and Bernard Chazelle. Approximate nearest neighbors and the fast johnson-lindenstrauss transform. In *Proceedings of the thirty-eighth annual ACM symposium on Theory of computing (STOC)*, pages 557–563, 2006.

[ACSS20] Josh Alman, Timothy Chu, Aaron Schild, and Zhao Song. Algorithms and hardness for linear algebra on geometric graphs. In *2020 IEEE 61st Annual Symposium on Foundations of Computer Science (FOCS)*, pages 541–552. IEEE, 2020.

[ACW16] Josh Alman, Timothy M Chan, and Ryan Williams. Polynomial representations of threshold functions and algorithmic applications. In *2016 IEEE 57th Annual Symposium on Foundations of Computer Science (FOCS)*, pages 467–476. IEEE, 2016.

[AIR18] Alexandr Andoni, Piotr Indyk, and Ilya Razenshteyn. Approximate nearest neighbor search in high dimensions. In *Proceedings of the International Congress of Mathematicians: Rio de Janeiro 2018*, pages 3287–3318. World Scientific, 2018.

[AKK+20] Thomas D Ahle, Michael Kapralov, Jakob BT Knudsen, Rasmus Pagh, Ameya Velingker, David P Woodruff, and Amir Zandieh. Oblivious sketching of high-degree polynomial kernels. In *Proceedings of the Fourteenth Annual ACM-SIAM Symposium on Discrete Algorithms*, pages 141–160. SIAM, 2020.

[AKR15] Alexandr Andoni, Robert Krauthgamer, and Ilya Razenshteyn. Sketching and embedding are equivalent for norms. In *Proceedings of the Forty-seventh Annual ACM Symposium on Theory of Computing (STOC)*, pages 479–488, 2015.

[Alm19]   Josh Alman. An illuminating algorithm for the light bulb problem. In *SOSA*. arXiv preprint arXiv:1810.06740, 2019.

[ALS+23]  Josh Alman, Jiehao Liang, Zhao Song, Ruizhe Zhang, and Danyang Zhuo. Bypass exponential time preprocessing: Fast neural network training via weight-data correlation preprocessing. In *NeurIPS*, 2023.

[Aro50]   Nachman Aronszajn. Theory of reproducing kernels. *Transactions of the American mathematical society*, 68(3):337–404, 1950.

[AS23]    Josh Alman and Zhao Song. Fast attention requires bounded entries. In *NeurIPS*, 2023.

[AS24a]   Josh Alman and Zhao Song. Fast rope attention: Combining the polynomial method and fast fourier transform. In *manuscript*, 2024.

[AS24b]   Josh Alman and Zhao Song. The fine-grained complexity of gradient computation for training large language models. In *NeurIPS*. arXiv preprint arXiv:2402.04497, 2024.

[AS24c]   Josh Alman and Zhao Song. How to capture higher-order correlations? generalizing matrix softmax attention to kronecker computation. In *ICLR*, 2024.

[Ass80]   Patrice Assouad. Plongements isométriques dans l1: aspect analytique. *Number*, 14:1979–1980, 1980.

[AW17]    Josh Alman and Ryan Williams. Probabilistic rank and matrix rigidity. In *Proceedings of the 49th Annual ACM SIGACT Symposium on Theory of Computing*, pages 641–652, 2017.

[AWY14]   Amir Abboud, Ryan Williams, and Huacheng Yu. More applications of the polynomial method to algorithm design. In *Proceedings of the twenty-sixth annual ACM-SIAM symposium on Discrete algorithms (SODA)*, pages 218–230. SIAM, 2014.

[Bai99]   René Baire. Sur les fonctions de variables réelles. *Annali di Matematica Pura ed Applicata (1898-1922)*, 3(1):1–123, 1899.

[BCR84]   Christian Berg, Jens Peter Reus Christensen, and Paul Ressel. *Harmonic analysis on semigroups: theory of positive definite and related functions*, volume 100. Springer, 1984.

[Ber29]   Serge Bernstein. Sur les fonctions absolument monotones. *Acta Mathematica*, 52(1):1–66, 1929.

[BMM18]   Mikhail Belkin, Siyuan Ma, and Soumik Mandal. To understand deep learning we need to understand kernel learning. In *International Conference on Machine Learning*, pages 541–549. PMLR, 2018.

[BMR+20]  Tom Brown, Benjamin Mann, Nick Ryder, Melanie Subbiah, Jared D Kaplan, Prafulla Dhariwal, Arvind Neelakantan, Pranav Shyam, Girish Sastry, Amanda Askell, et al. Language models are few-shot learners. *Advances in neural information processing systems (NeurIPS)*, 33:1877–1901, 2020.

[BPSW21]  Jan van den Brand, Binghui Peng, Zhao Song, and Omri Weinstein. Training (over-parametrized) neural networks in near-linear time. In *12th Innovations in Theoretical Computer Science Conference (ITCS)*, 2021.

[BTF04]   Sabri Boughorbel, Jean-Philippe Tarel, and Francois Fleuret. Non-mercer kernels for svm object recognition. In *BMVC*, pages 1–10, 2004.

[CDW+21]  Beidi Chen, Tri Dao, Eric Winsor, Zhao Song, Atri Rudra, and Christopher Ré. Scatterbrain: Unifying sparse and low-rank attention approximation. In *NeurIPS*, 2021.

[CFW12]   Ching-Shyang Chen, Chia-Ming Fan, and PH Wen. The method of approximate particular solutions for solving certain partial differential equations. *Numerical Methods for Partial Differential Equations*, 28(2):506–522, 2012.

[CHC+10]  Yin-Wen Chang, Cho-Jui Hsieh, Kai-Wei Chang, Michael Ringgaard, and Chih-Jen Lin. Training and testing low-degree polynomial data mappings via linear svm. *Journal of Machine Learning Research*, 11(4), 2010.

[CKPS16]  Xue Chen, Daniel M Kane, Eric Price, and Zhao Song. Fourier-sparse interpolation without a frequency gap. In *57th Annual Symposium on Foundations of Computer Science (FOCS)*, pages 741–750. IEEE, 2016.

[CLD+20]  Krzysztof Marcin Choromanski, Valerii Likhosherstov, David Dohan, Xingyou Song, Andreea Gane, Tamas Sarlos, Peter Hawkins, Jared Quincy Davis, Afroz Mohiuddin, Lukasz Kaiser, et al. Rethinking attention with performers. In *International Conference on Learning Representations (ICLR)*, 2020.

[CLP17]  Ernie Croot, Vsevolod F Lev, and Péter Pál Pach. Progression-free sets in are exponentially small. *Annals of Mathematics*, pages 331–337, 2017.

[CMS20]  Timothy Chu, Gary L. Miller, and Donald Sheehy. Exact computation of a manifold metric, via lipschitz embeddings and shortest paths on a graph. In *SODA*, 2020.

[CN21]  Yeshwanth Cherapanamjeri and Jelani Nelson. Terminal embeddings in sublinear time. In *FOCS*, 2021.

[CND+22]  Aakanksha Chowdhery, Sharan Narang, Jacob Devlin, Maarten Bosma, Gaurav Mishra, Adam Roberts, Paul Barham, Hyung Won Chung, Charles Sutton, Sebastian Gehrmann, et al. Palm: Scaling language modeling with pathways. *arXiv preprint arXiv:2204.02311*, 2022.

[CNW15]  Michael B Cohen, Jelani Nelson, and David P Woodruff. Optimal approximate matrix product in terms of stable rank. *arXiv preprint arXiv:1507.02268*, 2015.

[Cop82]  Don Coppersmith. Rapid multiplication of rectangular matrices. *SIAM Journal on Computing*, 11(3):467–471, 1982.

[Cri88]  Frank Critchley. On certain linear mappings between inner-product and squared-distance matrices. *Linear Algebra and its Applications*, 105:91–107, 1988.

[CUH15]  Djork-Arné Clevert, Thomas Unterthiner, and Sepp Hochreiter. Fast and accurate deep network learning by exponential linear units (elus). *arXiv preprint arXiv:1511.07289*, 2015.

[Cut09]  Marco Cuturi. Positive definite kernels in machine learning. *arXiv preprint arXiv:0911.5367*, 2009.

[CV95]  Corinna Cortes and Vladimir Vapnik. Support-vector networks. *Machine learning*, 20(3):273–297, 1995.

[CW16]  Timothy M Chan and Ryan Williams. Deterministic apsp, orthogonal vectors, and more: Quickly derandomizing razborov-smolensky. In *Proceedings of the twenty-seventh annual ACM-SIAM symposium on Discrete algorithms (SODA)*, pages 1246–1255. SIAM, 2016.

[DCLT18]  Jacob Devlin, Ming-Wei Chang, Kenton Lee, and Kristina Toutanova. Bert: Pre-training of deep bidirectional transformers for language understanding. *arXiv preprint arXiv:1810.04805*, 2018.

[DE19]  Zeev Dvir and Benjamin L Edelman. Matrix rigidity and the croot-lev-pach lemma. *Theory of Computing*, 15:1–7, 2019.

[Del73]  Philippe Delsarte. *An algebraic approach to the association schemes of coding theory*. PhD thesis, Philips Research Laboratories, 1973.

[DKS10]  Anirban Dasgupta, Ravi Kumar, and Tamás Sarlós. A sparse johnson: Lindenstrauss transform. In *Proceedings of the forty-second ACM symposium on Theory of computing*, pages 341–350, 2010.

[DL09]     Michel Marie Deza and Monique Laurent. *Geometry of Cuts and Metrics*. Springer Publishing Company, Incorporated, 1st edition, 2009.

[DL19]     Zeev Dvir and Allen Liu. Fourier and circulant matrices are not rigid. In *Computational Complexity Conference (CCC)*, volume 137, pages 17:1–17:23, 2019.

[DSWZ22]   Yichuan Deng, Zhao Song, Omri Weinstein, and Ruizhe Zhang. Fast distance oracles for any symmetric norm. *arXiv preprint arXiv:2205.14816*, 2022.

[EG17]     Jordan S Ellenberg and Dion Gijswijt. On large subsets of with no three-term arithmetic progression. *Annals of Mathematics*, pages 339–343, 2017.

[EGH+11]   Pavel I Etingof, Oleg Golberg, Sebastian Hensel, Tiankai Liu, Alex Schwendner, Dmitry Vaintrob, and Elena Yudovina. *Introduction to representation theory*, volume 59. American Mathematical Soc., 2011.

[FLH15]    Aasa Feragen, Francois Lauze, and Soren Hauberg. Geodesic exponential kernels: When curvature and linearity conflict. In *Proceedings of the IEEE conference on computer vision and pattern recognition (CVPR)*, pages 3032–3042, 2015.

[FLLA15]   Felix Faber, Alexander Lindmaa, O Anatole von Lilienfeld, and Rickard Armiento. Crystal structure representations for machine learning models of formation energies. *International Journal of Quantum Chemistry*, 115(16):1094–1101, 2015.

[Fro12]    Georg Frobenius. Über matrizen aus nicht negativen elementen. ., 1912.

[FS98]     Carsten Franke and Robert Schaback. Solving partial differential equations by collocation using radial basis functions. *Applied Mathematics and Computation*, 93(1):73–82, 1998.

[FS03]     François Fleuret and Hichem Sahbi. Scale-invariance of support vector machines based on the triangular kernel. In *3rd International Workshop on Statistical and Computational Theories of Vision*, pages 1–13, 2003.

[FZS21]    William Fedus, Barret Zoph, and Noam Shazeer. Switch transformers: Scaling to trillion parameter models with simple and efficient sparsity. *arXiv preprint arXiv:2101.03961*, 2021.

[GE08]     Yoav Goldberg and Michael Elhadad. splitsvm: fast, space-efficient, non-heuristic, polynomial kernel computation for nlp applications. In *Proceedings of ACL-08: HLT, Short Papers*, pages 237–240, 2008.

[GKR17]    Dominique Guillot, Apoorva Khare, and Bala Rajaratnam. Preserving positivity for rank-constrained matrices. *Transactions of the American Mathematical Society*, 369(9):6105–6145, 2017.

[GLS+24]   Jiuxiang Gu, Yingyu Liang, Zhizhou Sha, Zhenmei Shi, and Zhao Song. Differential privacy mechanisms in neural tangent kernel regression. *arXiv preprint arXiv:2407.13621*, 2024.

[GSDV17]   Aldo Glielmo, Peter Sollich, and Alessandro De Vita. Accurate interatomic force fields via machine learning with covariant kernels. *Physical Review B*, 95(21):214302, 2017.

[GSZ23]    Yuzhou Gu, Zhao Song, and Lichen Zhang. Faster algorithms for structured linear and kernel support vector machines. *arXiv preprint arXiv:2307.07735*, 2023.

[HAS20]    Insu Han, Haim Avron, and Jinwoo Shin. Polynomial tensor sketch for element-wise function of low-rank matrix. In *International Conference on Machine Learning*, pages 3984–3993. PMLR, 2020.

[HJK+23]   Insu Han, Rajesh Jarayam, Amin Karbasi, Vahab Mirrokni, David P Woodruff, and Amir Zandieh. Hyperattention: Long-context attention in near-linear time. *arXiv preprint arXiv:2310.05869*, 2023.

[HLSL24] Jerry Yao-Chieh Hu, Thomas Lin, Zhao Song, and Han Liu. On computational limits of modern hopfield models: A fine-grained complexity analysis. In *Forty-first International Conference on Machine Learning (ICML)*, 2024.

[HSK+24] Jerry Yao-Chieh Hu, Maojiang Su, En-Jui Kuo, Zhao Song, and Han Liu. Computational limits of low-rank adaptation (lora) for transformer-based models. *arXiv preprint arXiv:2406.03136*, 2024.

[HSM+00] Richard HR Hahnloser, Rahul Sarpeshkar, Misha A Mahowald, Rodney J Douglas, and H Sebastian Seung. Digital selection and analogue amplification coexist in a cortex-inspired silicon circuit. *Nature*, 405(6789):947–951, 2000.

[HSW+22] Baihe Huang, Zhao Song, Omri Weinstein, Hengjie Zhang, and Ruizhe Zhang. A dynamic fast gaussian transform. *arXiv preprint arXiv:2202.12329*, 2022.

[HTF09] T Hastie, R Tibshirani, and J Friedman. Data mining, inference, and prediction. the elements of statistical learning. ., 2009.

[HW78] A Hedayat and Walter Dennis Wallis. Hadamard matrices and their applications. *The Annals of Statistics*, 6(6):1184–1238, 1978.

[HWL+24] Jerry Yao-Chieh Hu, Weimin Wu, Zhuoru Li, Sophia Pi, , Zhao Song, and Han Liu. On statistical rates and provably efficient criteria of latent diffusion transformers (dits). In *Thirty-eighth Conference on Neural Information Processing Systems (NeurIPS)*, 2024.

[IMS17] Piotr Indyk, Jiří Matoušek, and Anastasios Sidiropoulos. low-distortion embeddings of finite metric spaces. In *Handbook of discrete and computational geometry*, pages 211–231. Chapman and Hall/CRC, 2017.

[JGH18] Arthur Jacot, Franck Gabriel, and Clément Hongler. Neural tangent kernel: Convergence and generalization in neural networks. In *Advances in neural information processing systems (NeurIPS)*, pages 8571–8580, 2018.

[KCC17] Irene Kaltenmark, Benjamin Charlier, and Nicolas Charon. A general framework for curve and surface comparison and registration with oriented varifolds. In *Proceedings of the IEEE conference on computer vision and pattern recognition (CVPR)*, pages 3346–3355, 2017.

[KKK18] Matti Karppa, Petteri Kaski, and Jukka Kohonen. A faster subquadratic algorithm for finding outlier correlations. *ACM Transactions on Algorithms (TALG)*, 14(3):1–26, 2018.

[KKL19] Nikita Kitaev, Lukasz Kaiser, and Anselm Levskaya. Reformer: The efficient transformer. In *International Conference on Learning Representations (ICLR)*, 2019.

[KM03] Taku Kudo and Yuji Matsumoto. Fast methods for kernel-based text analysis. In *Proceedings of the 41st Annual Meeting of the Association for Computational Linguistics*, pages 24–31, 2003.

[KMZ23] Praneeth Kacham, Vahab Mirrokni, and Peilin Zhong. Polysketchformer: Fast transformers via sketches for polynomial kernels. *arXiv preprint arXiv:2310.01655*, 2023.

[KUMH17] Günter Klambauer, Thomas Unterthiner, Andreas Mayr, and Sepp Hochreiter. Self-normalizing neural networks. *Advances in neural information processing systems*, 30, 2017.

[KVPF20] Angelos Katharopoulos, Apoorv Vyas, Nikolaos Pappas, and François Fleuret. Transformers are rnns: Fast autoregressive transformers with linear attention. In *International conference on machine learning*, pages 5156–5165. PMLR, 2020.

[KW71] George Kimeldorf and Grace Wahba. Some results on tchebycheffian spline functions. *Journal of mathematical analysis and applications*, 33(1):82–95, 1971.

[KZR16]  Soheil Kolouri, Yang Zou, and Gustavo K Rohde. Sliced wasserstein kernels for probability distributions. In *Proceedings of the IEEE Conference on Computer Vision and Pattern Recognition (CVPR)*, pages 5258–5267, 2016.

[Lax02]  Peter Lax. *Functional Analysis*. Wiley, 1st edition, 2002.

[Lil18]  O Anatole Von Lilienfeld. Quantum machine learning in chemical compound space. *Angewandte Chemie International Edition*, 57(16):4164–4169, 2018.

[LLLH18]  Shaogao Lv, Huazhen Lin, Heng Lian, and Jian Huang. Oracle inequalities for sparse additive quantile regression in reproducing kernel hilbert space. *The Annals of Statistics*, 46(2):781–813, 2018.

[LLSS24]  Chenyang Li, Yingyu Liang, Zhenmei Shi, and Zhao Song. Exploring the frontiers of softmax: Provable optimization, applications in diffusion model, and beyond. *arXiv preprint arXiv:2405.03251*, 2024.

[LNRW19]  Jerry Li, Aleksandar Nikolov, Ilya Razenshteyn, and Erik Waingarten. On mean estimation for general norms with statistical queries. In *Conference on Learning Theory*, pages 2158–2172. PMLR, 2019.

[LRRK15]  O Anatole Von Lilienfeld, Raghunathan Ramakrishnan, Matthias Rupp, and Aaron Knoll. Fourier series of atomic radial distribution functions: A molecular fingerprint for machine learning models of quantum chemical properties. *International Journal of Quantum Chemistry*, 115(16):1084–1093, 2015.

[LSS+20]  Jason D Lee, Ruoqi Shen, Zhao Song, Mengdi Wang, and Zheng Yu. Generalized leverage score sampling for neural networks. In *NeurIPS*, 2020.

[LSS+24]  Yingyu Liang, Zhizhou Sha, Zhenmei Shi, Zhao Song, and Yufa Zhou. Multi-layer transformers gradient can be approximated in almost linear time. *arXiv preprint arXiv:2408.13233*, 2024.

[LSSZ24a]  Yingyu Liang, Zhenmei Shi, Zhao Song, and Yufa Zhou. Differential privacy of cross-attention with provable guarantee. *arXiv preprint arXiv:2407.14717*, 2024.

[LSSZ24b]  Yingyu Liang, Zhenmei Shi, Zhao Song, and Yufa Zhou. Tensor attention training: Provably efficient learning of higher-order transformers. *arXiv preprint arXiv:2405.16411*, 2024.

[LSZ+19]  Xuanqing Liu, Si Si, Xiaojin Zhu, Yang Li, and Cho-Jui Hsieh. A unified framework for data poisoning attack to graph-based semi-supervised learning. In *Advances in Neural Information Processing Systems (NeurIPS)*, 2019.

[LWD+23]  Zichang Liu, Jue Wang, Tri Dao, Tianyi Zhou, Binhang Yuan, Zhao Song, Anshumali Shrivastava, Ce Zhang, Yuandong Tian, Christopher Re, et al. Deja vu: Contextual sparsity for efficient llms at inference time. In *International Conference on Machine Learning*, pages 22137–22176. PMLR, 2023.

[Mer09]  James Mercer. Functions of positive and negative type, and their connection the theory of integral equations. *Philosophical transactions of the royal society of London. Series A, containing papers of a mathematical or physical character*, 209(441-458):415–446, 1909.

[MMMR18]  Sepideh Mahabadi, Konstantin Makarychev, Yury Makarychev, and Ilya Razenshteyn. Nonlinear dimension reduction via outer bi-lipschitz extensions. In *Proceedings of the 50th Annual ACM SIGACT Symposium on Theory of Computing (STOC)*, pages 1088–1101, 2018.

[MMR19]  Konstantin Makarychev, Yury Makarychev, and Ilya Razenshteyn. Performance of johnson-lindenstrauss transform for k-means and k-medians clustering. In *Proceedings of the 51st Annual ACM SIGACT Symposium on Theory of Computing (STOC)*, pages 1027–1038, 2019.

[MOSW22] Alexander Munteanu, Simon Omlor, Zhao Song, and David P. Woodruff. Bounding the width of neural networks via coupled initialization A worst case analysis. In *ICML*, volume 162 of *Proceedings of Machine Learning Research*, pages 16083–16122. PMLR, 2022.

[MPS16] Daniel Moeller, Ramamohan Paturi, and Stefan Schneider. Subquadratic algorithms for succinct stable matching. In *International Computer Science Symposium in Russia*, pages 294–308. Springer, 2016.

[MSS17] Adam W Marcus, Daniel A Spielman, and Nikhil Srivastava. Interlacing families iii: Sharper restricted invertibility estimates. *arXiv preprint arXiv:1712.07766*, 2017.

[NJW02] Andrew Y Ng, Michael I Jordan, and Yair Weiss. On spectral clustering: Analysis and an algorithm. In *Advances in neural information processing systems (NeurIPS)*, pages 849–856, 2002.

[NN19] Shyam Narayanan and Jelani Nelson. Optimal terminal dimensionality reduction in euclidean space. In *Proceedings of the 51st Annual ACM SIGACT Symposium on Theory of Computing (STOC)*, pages 1064–1069, 2019.

[NS41] J. Von Neumann and I. J. Schoenberg. Fourier integrals and metric geometry. *Transactions of the American Mathematical Society*, 50(2):226–251, 1941.

[O'D14] Ryan O'Donnell. *Analysis of Boolean Functions*. Cambridge University Press, New York, NY, USA, 2014.

[Orr96] Mark JL Orr. Introduction to radial basis function networks, 1996.

[Par12] Pablo A. Parrilo. Algebraic techniques and semidefinite optimization, February 2012.

[Per07] Oskar Perron. Zur theorie der matrices. *Mathematische Annalen*, 64(2):248–263, 1907.

[RNS⁺18] Alec Radford, Karthik Narasimhan, Tim Salimans, Ilya Sutskever, et al. Improving language understanding by generative pre-training. 2018.

[RR08] Ali Rahimi and Benjamin Recht. Random features for large-scale kernel machines. In *Advances in neural information processing systems (NIPS)*, pages 1177–1184, 2008.

[RSZ22] Aravind Reddy, Zhao Song, and Lichen Zhang. Dynamic tensor product regression. In *Conference on Neural Information Processing Systems (NeurIPS)*, pages 4791–4804, 2022.

[RWC⁺19] Alec Radford, Jeffrey Wu, Rewon Child, David Luan, Dario Amodei, Ilya Sutskever, et al. Language models are unsupervised multitask learners. *OpenAI blog*, 1(8):9, 2019.

[RYW⁺19] Emily Reif, Ann Yuan, Martin Wattenberg, Fernanda B Viegas, Andy Coenen, Adam Pearce, and Been Kim. Visualizing and measuring the geometry of bert. *Advances in Neural Information Processing Systems*, 32, 2019.

[Sch35] I. J. Schoenberg. Remarks to maurice frechet's article" sur la definition axiomatique d'une classe d'espace distancies vector! ellement applicable sur l'espace de hilbertl. *Ann. of Math*, 36:724–732, 1935.

[Sch37] I. J. Schoenberg. On certain metric spaces arising from euclidean spaces by a change of metric and their imbedding in hilbert space. *Annals of Mathematics*, 38(4):787–793, 1937.

[Sch38] I. J. Schoenberg. Metric spaces and positive definite functions. *Trans. Amer. Math. Soc.*, 44:522–536, 1938.

[Sch42] I Schoenberg. Positive definite functions on spheres. *Duke Math. J*, 1:172, 1942.

[Smo96] Alex J Smola. *Regression estimation with support vector learning machines*. PhD thesis, Master's thesis, Technische Universität München, 1996.

[Sou10] César R Souza. Kernel functions for machine learning applications. *Creative Commons Attribution-Noncommercial-Share Alike*, 3:29, 2010.

[SOW01] Alex J Smola, Zoltan L Ovari, and Robert C Williamson. Regularization with dot-product kernels. In *Advances in neural information processing systems (NeurIPS)*, pages 308–314, 2001.

[SS01] Bernhard Scholkopf and Alexander J Smola. *Learning with kernels: support vector machines, regularization, optimization, and beyond*. MIT press, 2001.

[SSB+97] Bernhard Scholkopf, Kah-Kay Sung, Christopher JC Burges, Federico Girosi, Partha Niyogi, Tomaso Poggio, and Vladimir Vapnik. Comparing support vector machines with gaussian kernels to radial basis function classifiers. *IEEE transactions on Signal Processing*, 45(11):2758–2765, 1997.

[SSL24] Yiyou Sun, Zhenmei Shi, and Yixuan Li. A graph-theoretic framework for understanding open-world semi-supervised learning. *Advances in Neural Information Processing Systems*, 36, 2024.

[SSLL23] Yiyou Sun, Zhenmei Shi, Yingyu Liang, and Yixuan Li. When and how does known class help discover unknown ones? provable understanding through spectral analysis. In *International Conference on Machine Learning*, pages 33014–33043. PMLR, 2023.

[SSM98] Alex J Smola, Bernhard Schölkopf, and Klaus-Robert Müller. The connection between regularization operators and support vector kernels. *Neural networks*, 11(4):637–649, 1998.

[SSWZ23a] Tamas Sarlos, Xingyou Song, David Woodruff, and Richard Zhang. Hardness of low rank approximation of entrywise transformed matrix products. *Advances in Neural Information Processing Systems*, 36, 2023.

[SSWZ23b] Zhao Song, Baocheng Sun, Omri Weinstein, and Ruizhe Zhang. Quartic samples suffice for fourier interpolation. In *FOCS*, 2023.

[SWYZ21] Zhao Song, David P. Woodruff, Zheng Yu, and Lichen Zhang. Fast sketching of polynomial kernels of polynomial degree. In *ICML*, 2021.

[SYZ21] Zhao Song, Shuo Yang, and Ruizhe Zhang. Does preprocessing help training over-parameterized neural networks? *Advances in Neural Information Processing Systems*, 34, 2021.

[SYZ24] Zhao Song, Junze Yin, and Lichen Zhang. Solving attention kernel regression problem via pre-conditioner. In *International Conference on Artificial Intelligence and Statistics (AISTATS)*, pages 208–216. PMLR, 2024.

[SZZ24] Zhao Song, Lichen Zhang, and Ruizhe Zhang. Training multi-layer over-parametrized neural network in subquadratic time. In *ITCS*, 2024.

[TBY+19] Yao-Hung Hubert Tsai, Shaojie Bai, Makoto Yamada, Louis-Philippe Morency, and Ruslan Salakhutdinov. Transformer dissection: An unified understanding for transformer's attention via the lens of kernel. In Kentaro Inui, Jing Jiang, Vincent Ng, and Xiaojun Wan, editors, *Proceedings of the 2019 Conference on Empirical Methods in Natural Language Processing and the 9th International Joint Conference on Natural Language Processing, EMNLP-IJCNLP 2019, Hong Kong, China, November 3-7, 2019*, pages 4343–4352. Association for Computational Linguistics, 2019.

[Val12] Gregory Valiant. Finding correlations in subquadratic time, with applications to learning parities and juntas. In *2012 IEEE 53rd Annual Symposium on Foundations of Computer Science (FOCS)*, pages 11–20, 2012.

[vL07] Ulrike von Luxburg. A tutorial on spectral clustering, 2007.

[VSP+17] Ashish Vaswani, Noam Shazeer, Niki Parmar, Jakob Uszkoreit, Llion Jones, Aidan N Gomez, Łukasz Kaiser, and Illia Polosukhin. Attention is all you need. *Advances in neural information processing systems (NeurIPS)*, 30, 2017.

[Wil18]    R Ryan Williams. Faster all-pairs shortest paths via circuit complexity. *SIAM Journal on Computing*, 47(5):1965–1985, 2018.

[WLK⁺20]   Sinong Wang, Belinda Z Li, Madian Khabsa, Han Fang, and Hao Ma. Linformer: Self-attention with linear complexity. *arXiv preprint arXiv:2006.04768*, 2020.

[WZF05]    Liwei Wang, Yan Zhang, and Jufu Feng. On the euclidean distance of images. *IEEE transactions on pattern analysis and machine intelligence*, 27(8):1334–1339, 2005.

[XZZ18]    Chang Xiao, Peilin Zhong, and Changxi Zheng. Bourgan: generative networks with metric embeddings. In *Proceedings of the 32nd International Conference on Neural Information Processing Systems (NeurIPS)*, pages 2275–2286, 2018.

[Zha22]    Lichen Zhang. *Speeding up optimizations via data structures: Faster search, sample and maintenance*. PhD thesis, Master's thesis, Carnegie Mellon University, 2022.

[ZHDK23]   Amir Zandieh, Insu Han, Majid Daliri, and Amin Karbasi. Kdeformer: Accelerating transformers via kernel density estimation. In *ICML*. arXiv preprint arXiv:2302.02451, 2023.

[Zhu05a]   Xiaojin Zhu. *Semi-supervised learning with graphs*. PhD thesis, Carnegie Mellon University, language technologies institute, school of Computer Science, 2005.

[Zhu05b]   Xiaojin Jerry Zhu. Semi-supervised learning literature survey. Technical report, University of Wisconsin-Madison Department of Computer Sciences, 2005.

[ZLZ24]    Shijun Zhang, Jianfeng Lu, and Hongkai Zhao. Deep network approximation: Beyond relu to diverse activation functions. *Journal of Machine Learning Research*, 25(35):1–39, 2024.

[ZRG⁺22]   Susan Zhang, Stephen Roller, Naman Goyal, Mikel Artetxe, Moya Chen, Shuohui Chen, Christopher Dewan, Mona Diab, Xian Li, Xi Victoria Lin, et al. Opt: Open pre-trained transformer language models. *arXiv preprint arXiv:2205.01068*, 2022.

[ZSZ⁺23]   Zhenyu Zhang, Ying Sheng, Tianyi Zhou, Tianlong Chen, Lianmin Zheng, Ruisi Cai, Zhao Song, Yuandong Tian, Christopher Ré, Clark Barrett, Zhangyang Wang, and Beidi Chen. H _2 o: Heavy-hitter oracle for efficient generative inference of large language models. In *NeurIPS*. arXiv preprint arXiv:2306.14048, 2023.

[ZZP⁺21]   Haoyi Zhou, Shanghang Zhang, Jieqi Peng, Shuai Zhang, Jianxin Li, Hui Xiong, and Wancai Zhang. Informer: Beyond efficient transformer for long sequence time-series forecasting. In *Proceedings of the AAAI conference on artificial intelligence*, volume 35, pages 11106–11115, 2021.

# Appendix

# Contents

**Roadmap.** In Section A, we define notations, and provide several basic definitions and fundamental tools. In Section B, we provide several techniques used to prove the theorems. In Section C, we prove the converse to the polynomial method, proving Theorem 2.5. In Section D, we prove an approximate converse to the polynomial method, proving Theorem 2.6. In Section E, we prove condition 1 and condition 2 in Theorem 4.4 on Manhattan metric transforms are equivalent. In Section F, we prove condition 2 and condition 3 in Theorem 4.4 are equivalent. Overall, Section E and Section F together prove Theorem 4.4. In Section G, we have a proof of Theorem 3.4 about Manhattan distance kernels. In Section H, we have a new proof for the known, full classification of Euclidean distance kernels. In Section I, we prove a slightly different restatement of Lemma B.1. We show our results about stable rank in Section J.

## A Preliminaries

This section is organized as follows:

- In Section A.1, we define several basic notations.
- In Section A.2, we provide some definitions piecewise functions, open/closed/dense sets, real analytic functions, and finite differences.
- In Section A.3, we provide some previous work about the classifications of completely monotone and Bernstein function.
- In Section A.4, we state well-known results about metric hierarchies.
- In Section A.5, we define negative metrics and euclidean embeddability.
- In Section A.6, we present previous work about representation theory tools.
- In Section A.7, we present the Baire category theorem.
- In Section A.8, we discuss some applications of polynomial methods.

### A.1 Notations

For a vector $x$, we use $\|x\|_1$ to denote the entry-wise $\ell_1$ norm of $x$. We use $\|x\|_2$ to denote the entry-wise $\ell_2$ norm of $x$. We use $\|x\|_\infty$ to denote the $\ell_\infty$ norm of $x$. For two vectors $a, b$, we use $\langle a, b \rangle$ to denote the inner product between $a$ and $b$. For a vector $x$, we use $x^\top$ to denote the transpose of $x$. For any matrix $A$, we use $\lambda_i$'s to denote its eigenvalues. For any square matrix $A$, we use $\det(A)$ to denote its determinant.

For any $d \geq 1$, we define Hadamard matrix $H_d \in \mathbb{R}^{2^d \times 2^d}$ as follows $H_d = \begin{bmatrix} H_{d-1} & H_{d-1} \\ H_{d-1} & -H_{d-1} \end{bmatrix}$ and $H_0 = 1$.

Often times in our proof, we may say things like "let $x_1, \ldots x_{2^d}$ be the corners of a $d$ dimensional hyperrectangle". For these statements to make sense, we must specify which corner $x_i$ refers to. Scale the $d$ dimensional hyperrectangle to be an axis-aligned hypercube, and place one of the hypercube corners at the origin. Each corner then has a binary number $b$ as its coordinate bit string. We let $x_{b+1}$ refer to the original hyperrectangle corner corresponding to $b$.

### A.2 Definitions

**Piecewise function** A piecewise function is a function defined by multiple sub-functions, where each sub-function applies to a different interval in the domain.

**Open, closed and dense set** Open sets are a generalization of open intervals in the real line. In a metric space (with a pre-defined distance function) open sets are the sets that, with every point $P$, contain all points that are sufficiently near to $P$ (that is, all points whose distance to $P$ is less than some value depending on $P$).

A closed set is a set whose complement is an open set. A set that is closed under an operation or collection of operations is said to satisfy a closure property.

A subset $A$ of a topological space $X$ is called dense (in $X$) if every point $x$ in $X$ either belongs to $A$ or is a limit point of $A$; that is, the closure of $A$ constitutes the whole set $X$.

The interior of a subset $S$ of a topological space $X$ is the union of all subsets of $S$ that are open in $X$.

**Real Analytic**  Formally, a function $f$ is real analytic on an open set $D$ in the real line if for any $x_0 \in D$ one can write

$$f(x) = \sum_{n=0}^{\infty} a_n (x - x_0)^n = a_0 + a_1(x - x_0) + a_2(x - x_0)^2 + a_3(x - x_0)^3 + \cdots$$

in which the coefficients $a_0, a_1, \cdots$ are real numbers and the series is convergent to $f(x)$ for $x$ in a neighborhood of $x_0$.

The following conditions are equivalent:

- $f$ is real analytic on an open set $D$.
- There is a complex analytic extension of $f$ to an open set $G \subset \mathbb{C}$ which contains $D$.
- $f$ is real smooth and for every compact set $K \subset D$ there exists a constant $C$ such that for every $x \in K$ and every non-negative integer $k$ the following bound holds $|\frac{\mathrm{d}^k f}{\mathrm{d} x^k}(x)| \leq C^{k+1} k!$.

**Finite Difference**  A finite difference is a mathematical expression of the form $f(x + b) - f(x + a)$.

Three basic types are commonly considered: forward, backward, and central finite differences.

A forward difference, denoted $\Delta_h[f]$, of a function $f$ is a function defined as

$$\Delta_h[f](x) = f(x + h) - f(x).$$

A backward difference uses the function values at $x$ and $x - h$, instead of the values at $x + h$ and $x$:

$$\nabla_h[f](x) = f(x) - f(x - h) = \Delta_h[f](x - h).$$

Finally, the central difference is given by

$$\delta_h[f](x) = f(x + h/2) - f(x - h/2) = \Delta_h[f](x - h/2).$$

In this paper, we will mainly focus on foward difference.

We use $\Delta_h^d[f](x)$ to denote $d$-th difference for any $h \in \mathbb{R}^d$ at $x \in \mathbb{R}$. For each $i \in [d]$, we use $h_i$ to denote the $i$-th entry of $h$. For each $j \in [d]$, the $j$-th finite difference can be written as the following recursive way. For $j = 1$, we have

$$\Delta_{h_1}^2[f](x) = f(x + h_1) - f(x).$$

For $j = 2$, we have

$$\Delta_{h_1, h_2}^2[f](x) = \Delta_{h_1}[f](x + h_2) - \Delta_{h_1}[f](x).$$

For each $j \in [d]$, we have

$$\Delta_{h_1, h_2, \cdots, h_j}^j[f](x) = \Delta_{h_1, h_2, \cdots h_{j-1}}^{j-1}[f](x + h_j) - \Delta_{h_1, h_2, \cdots h_{j-1}}^{j-1}[f](x).$$

We will frequently apply these finite differences to functions $g : \mathbb{R}^d \to \mathbb{R}$ of the form $g(a) = f(\langle a, 1 \rangle)$ for a function $f : \mathbb{R} \to \mathbb{R}$ or similar. In these cases, we will abuse notation and write $\Delta_\varepsilon^d[f](\langle a, 1 \rangle)$ to refer to $\Delta_\varepsilon^d[g](a)$.

### A.3  Alternate Classifications of Completely Monotone and Bernstein Functions

Here we recall the classical Bernstein Theorem from analysis constructively classifying completely monotone (Definition 3.2) and Bernstein functions (Definition 4.3).

**Proposition A.1** (Chapter 14, Theorems 3 and 6 in [Lax02])**.**  *For a function $f : \mathbb{R}_{>0} \to \mathbb{R}_{\geq 0}$, the following are equivalent:*

1. *f is completely monotone.*

2. *Letting $(D_a f)(x) = f(x+a) - f(x)$, for any $(a_1, \ldots, a_n)$ non-negative we have*

$$(-1)^n \left( \prod_{i=1}^{n} D_{a_i} \right) f(x) \geq 0$$

   *for all $x > 0$.*

3. *There exists a positive finite measure $\mu$ on $\mathbb{R}_{\geq 0}$ such that*

$$f(x) = \int_0^\infty e^{-tx} \mathrm{d}\mu(t), \quad x > 0.$$

The part 2 of Proposition A.1 is essentially the definition we gave for completely monotone, except that it does not assume any smoothness or even continuity a priori. The third shows that all completely monotone functions are in fact mixtures of decaying exponentials. From the above one easily derives a corresponding classification of Bernstein functions. If $f$ also has 0 in its domain, then the above result applies the same way, however (with the same measure $\mu$ as in part 3 of Proposition A.1) we have

$$f(0) \geq \mu(\mathbb{R}_{\geq 0})$$

since we did not require any continuity at 0.

**Proposition A.2** (Theorem 6.7 in [BCR84]). *For a function $f : \mathbb{R}_{\geq 0} \to \mathbb{R}_{\geq 0}$ with $f(0) = 0$, the following are equivalent:*

1. *f is Bernstein.*

2. *Letting $(D_a f)(x) = f(x+a) - f(x)$, for any $(a_1, \ldots, a_n)$ non-negative we have*

$$(-1)^n \left( \prod_{i=1}^{n} D_{a_i} \right) f(x) \leq 0, \quad x > 0.$$

3. *There exists a positive measure $\mu$ on $\mathbb{R}^+$ and $a, b \geq 0$ such that*

$$f(x) = a + bx + \int_{\mathbb{R}^+} (1 - e^{-tx}) \mathrm{d}\mu(t), \quad x > 0.$$

   *Here $\mu$ must satisfy $\int_{\mathbb{R}_+} \min\{1, t\} \mathrm{d}\mu(t) < \infty$.*

Due to the second criterion just above, Bernstein functions are also sometimes called *completely alternating*. We remark that these results apply more generally in the setting of abelian semigroups, where the integral is taken over a measure on the space of positive characters. This general point of view is explained in [BCR84, Chapter 6], and applies, for instance, to the semigroup of compact subsets of $\mathbb{R}$ under union.

## A.4  Metric Hierarchies

Here are well-known facts we will use throughout our proof:

**Lemma A.3.** *For any $n$ points $x_1, \ldots x_n$ in $\ell_1$, there exist $n$ points $y_1, \ldots y_n$ such that $\|x_i - x_j\|_1 = \|y_i - y_j\|_1$, and $y_1, \ldots y_n$ are a subset of corners of a $d$ dimensional hyperrectangle for some $d$.*

*Proof.* This follows from the equivalence of the cut cone and $\ell_1$ distance (Theorem 4.2.2 in [DL09]). $\square$

**Lemma A.4.** *The squared Euclidean distance between points in the corners of a hyperrectangle isometrically embeds into Manhattan distance.*

*Proof.* This follows from the Pythagorean theorem. □

**Lemma A.5.** *Manhattan distances embed isometrically into squared Euclidean distances.*

*Proof.* This follows from Corollary 6.1.4 and Lemma 6.1.7 in [DL09]. □

## A.5 Negative Type Metrics and Euclidean Embeddability

We now present a criterion by Schoenberg [Sch35] on when a metric is isometrically embeddable into squared Euclidean distances[7].

**Definition A.6** (negative type). *A matrix $D$ is iff $x^\top D x \leq 0$ for all $x \perp 1$.*

**Lemma A.7** (Schoenberg [Sch35]). *Consider $x_1, \ldots, x_n$ where $d_{i,j}$ is the distance between $x_i$ and $x_j$. Let $D$ be an $n$ by $n$ matrix where $D_{i,j} = d_{i,j}^2$. The distances $d_{i,j}$ are isometrically embeddable into Euclidean space iff the matrix $D$ is negative type.*

We note that if $D$ happens to have the all ones vector $\mathbf{1}$ as an eigenvector, we have a simpler criterion for testing if $D$ is negative type:

**Lemma A.8** (Schoenberg Variant). *Consider $x_1, \ldots, x_n$ where $d_{i,j}$ is the distance between $x_i$ and $x_j$. Let $D$ be an $n$ by $n$ matrix where $D_{i,j} = d_{i,j}^2$.*

*If the all ones vector is an eigenvector of $D$, then the $d_{i,j}$ are isometrically embeddable into Euclidean space iff every eigenvalue of $D$, excluding the eigenvalue correseponding to the all ones vector, is non-positive.*

*Proof.* Lemma A.8 follows from Lemma A.7 and the fact that every symmetric matrix has an orthonormal set of eigenvectors. □

If $d_{ij}$ is isometrically embeddable into Euclidean space, we can find an explicit embedding:

**Lemma A.9.** *Consider $x_1, \ldots x_n$ where $d_{i,j}$ is the distance between $x_i$ and $x_j$. Let $D$ be the matrix where $D_{i,j} = d_{i,j}^2$. Let $\Pi$ be the projection matrix off the all ones vector, i.e., $\Pi$ can be expressed explicitly as $I - J/n$, where $J$ is the $n \times n$ all-ones matrix, and $I$ is identity matrix.*

*Let $M := -\frac{1}{2}\Pi D \Pi$.*

*If $y_1, \ldots y_n$ are such that $\|y_i - y_j\|_2 = d_{i,j}$ and $\sum_{i=1}^n y_i = 0$, then $M_{i,j} = \langle y_i, y_j \rangle$. Moreover, if $M = U^\top U$ for some $U$, then the columns of $U$ are an embedding of $x_1, \ldots x_n$ into Euclidean space.*

This follows from Eq. 2 in [Cri88]. A longer exposition of the link between distance matrices and inner product matrices can be found in [Cri88].

## A.6 Schur's Lemma for Abelian Groups

We present Schur's lemma for Abelian groups $G$. Schur's lemma is one of the cornerstones of representation theory [EGH+11].

**Lemma A.10** (Schur's lemma for Abelian groups). *If $G$ is a finite Abelian group of $n \times n$ matrices under multiplication, and $M$ is an $n \times n$ diagonalizable matrix satisfying $Mg = gM$, for all $g \in G$, then there exists a set of linearly independent vectors $v_1, \ldots v_n$ that are eigenvectors of $M$ and all $g \in G$. In other words, $M$ and $G$ are simultaneously diagonalizable.*

Schur's Lemma will be useful in proving our key result about representation theory of the real hyperrectangle, or Lemma B.1.

## A.7 Baire Category Theorem

The Baire category theorem (BCT) is an important result in general topology and functional analysis.

A Baire space is a topological space with the property that for each countable collection of open dense sets $(U_n)_{n=1}^\infty$, their intersection $\bigcap_{n \in \mathbb{N}} U_n$ is dense.

**Theorem A.11** (Baire category theorem [Bai99]). *Every complete pseudometric space is a Baire space. Every locally compact Hausdorff space is a Baire space.*

## A.8 Applications of Polynomial Methods

**Attention Computation:** Question 2.3 is also important to the theory of polynomial kernels. A common computational task which arises when training transformers is to calculate the 'self attention' [VSP+17]. Recently, [ZHDK23, AS23] define a formal math computation problem for this attention mechanism. Formally, in this task, we are given three matrices $Q, K, V \in \mathbb{R}^{n \times d}$ where $n \gg d$,[8] and we would like to compute $(QK^\top)^f \cdot V$ where $f : \mathbb{R} \to \mathbb{R}$ is a non-linear function that we apply entry-wise to the matrix $AB^\top \in \mathbb{R}^{n \times n}$, then we multiply the result on the right by $V$. In many applications, $f$ is the soft-max function, which can be mathematically described as follows (see [ZHDK23, AS23]):

$$(QK^\top)^f := D^{-1} \exp(QK^\top), \quad \text{where} \quad D := \mathrm{diag}(\exp(QK^\top)\mathbf{1}_n)$$

Here $\exp()$ is an entry-wise function that $\exp(QK^\top)_{i,j} = \exp((QK^\top)_{i,j})$ for all $i, j \in [n] \times [n]$, $D \in \mathbb{R}^{n \times n}$ is a diagonal matrix, $\mathbf{1}_n$ is a vector that all entries are ones.

Naively evaluating $(QK^\top)^f \cdot V$ takes time $O(n^2 d)$ (without using fast matrix multiplication). However, if we can quickly find matrices $\widetilde{Q}, \widetilde{K} \in \mathbb{R}^{n \times \widetilde{d}}$ for some $\widetilde{d} < n$ such that $(QK^\top)^f = \widetilde{Q} \times \widetilde{K}^\top$, then we can evaluate it more quickly by first computing $\widetilde{K}^\top \times V$ and then computing $\widetilde{Q} \times (\widetilde{K}^\top \times V)$, for a total running time of just $O(nd\widetilde{d})$.

Since $QK^\top$ can be any rank $d$ matrix, and $\widetilde{Q} \times \widetilde{K}^\top$ has rank at most $\widetilde{d}$, it follows that an upper bound on the best $\widetilde{d}$ we can achieve is the maximum, over all matrices $M$ of rank $d$, of $\mathrm{rank}(M^f)$. Question 2.3 asks whether it is possible to achieve $d' < n$ for functions $f$ like the soft-max function which are not a polynomial. If not, then we can only hope to carry out this plan of attack if we can find a low-degree polynomial approximation to our function $f$.

**Algorithm Design:**

For one example of polynomial method in algorithm design, consider the fastest known algorithm for batch Hamming Nearest Neighbor Search due to Alman, Chan, and Williams [ACW16]. In this problem, one is given as input $2n$ vectors

$$x_1, \ldots, x_n, y_1, \ldots, y_n \in \{0, 1\}^d$$

for $d = \Theta(\log n)$, and a threshold value $t \in \{0, 1, \ldots, d\}$, and one wants to find a pair $(i, j) \in [n] \times [n]$ such that the Hamming distance between $x_i$ and $y_j$ is at most $t$. [ACW16] takes an algebraic approach to this problem, by first considering the matrix $M \in \mathbb{R}^{n \times n}$ where $M_{i,j}$ is the Hamming distance between $x_i$ and $y_j$. One can see that $\mathrm{rank}(M) \le 2d$, and one could use fast matrix multiplication to quickly compute all the entries of $M$.[9] However, since $M$ itself has $n^2$ entries, this could not improve much on the straightforward $O(n^2 \log n)$ time algorithm. They instead take the following approach.

First, pick a parameter $g = n^\delta$ for a constant $\delta > 0$, and a function $f : \mathbb{R} \to \mathbb{R}$ such that $f(x) > g^2$ for all $x \in \{0, 1, \ldots, t\}$, and $f(x) \in [0, 1]$ for all $x \in \{t+1, t+2, \ldots, d\}$. [ACW16] use Chebyshev polynomials to construct such an $f$ which is a low-degree polynomial, so that the matrix $M^f$ has low rank by Fact 2.1. Next, let $S_1, \ldots, S_{n/g}$ be a partition of $[n]$ into $n/g$ groups of size $g$, and consider the matrix $F \in \mathbb{R}^{\frac{n}{g} \times \frac{n}{g}}$ given by

$$F_{a,b} = \sum_{i \in S_a} \sum_{j \in S_b} M_{i,j}^f.$$

It is not hard to verify that $\text{rank}(F) \leq \text{rank}(M^f)$. Moreover, by the way $f$ was defined, an entry $F_{a,b}$ is larger than $g^2$ if and only if there is an $(i, j) \in S_a \times S_b$ such that the Hamming distance between $x_i$ and $y_j$ is at most $t$.

There is a trade-off between the parameter $\delta$ and the degree of $f$, and hence the rank of $F$. [ACW16] balance this trade-off to yield a matrix $F$ of low rank[10] and dimensions $n^{1-\delta} \times n^{1-\delta}$ for some $\delta > 0$. Since $F$ now has a subquadratic total number of entries, fast matrix multiplication can be used to compute all its entries and solve the problem, in roughly $O(n^{2-2\delta})$ time.

## B   Technique Overview

In this section, we describe the techniques used to prove our main theorems. In Section B.1, we introduce the eigenvalue properties of kernel matrices derived from hyperrectangles. In Section B.2, we introduce the techniques used to prove Theorem 2.5 and 2.6. In Section B.3, we introduce techniques used to prove Theorem 4.4. In Section B.4, we introduce techniques used to prove Theorem 3.4.

### B.1   Starting Point: Eigenvalues of the Kernel Matrix of a Hyperrectangle

All of our proofs start by using a simple but powerful technique. This technique computes eigenvectors and eigenvalues of the kernel matrix for any set of points which arise as the vertices of a hyperrectangle ($d$-dimensional rectangle). After describing the technique in more detail, we will explain how it leads to our applications by demonstrating why these matrices and their eigenvalues are relevant to the three main questions we stated in Section 1.

The eigenvectors of the family of matrices we define shortly will come from columns of Walsh-Hadamard matrices. For a positive integer $d$, let $v_1, \ldots v_{2^d} \in \{0, 1\}^d$ be the enumeration of all $n$-bit vectors in lexicographical order. The *Walsh-Hadamard* matrix $H_d$ is the $2^d \times 2^d$ matrix defined by $H_d(v_i, v_j) := (-1)^{\langle v_i, v_j \rangle}$. The technique is as follows:

**Lemma B.1** (Eigenvalue of Manhattan Kernels, informal version of Lemma I.2). *For a vector $a \in \mathbb{R}_{>0}^d$, let $p_1, \ldots, p_{2^d} \in \mathbb{R}^d$ denote the vertices $(\pm a_1/2, \pm a_2/2, \ldots, \pm a_d/2)$ of a hyperrectangle in lexicographical order. For any $f : \mathbb{R} \to \mathbb{R}$, let $D$ be the $2^d$ by $2^d$ matrix given by $D_{i,j} = f(\|p_i - p_j\|_1)$. Then, the columns of the Hadamard matrix $H_n$ are the eigenvectors of $D$.*

*For $i \in [2^d]$, let $B(i) \in \{0, 1\}^d$ be the binary representation of $i$. Then, the eigenvalue corresponding to column $i$ of $H_n$ is: $\lambda_i = \sum_{b \in \{0,1\}^d} (-1)^{\langle B(i), b \rangle} \cdot f(\langle b, a \rangle)$.*

We will see shortly that this expression for the eigenvalue $\lambda_i$ can also be rewritten in terms of integrals and derivatives of the function $f$, allowing us to use analytic techniques when computing or applying these eigenvalues.

Lemma B.1 can be proved using a direct calculation, although its inspiration comes from representation theory. The matrix $D$ has the property that: for any permutation matrix $\sigma$ corresponding to a reflection about one of the hyperrectangle's axes, we have $\sigma D = D\sigma$. Schur's lemma from representation theory (see Lemma A.10 below) states that $D$ and all $\sigma$ in the reflectional symmetry group of the hyperrectangle have a common set of eigenvectors. It is not hard to verify that the only common set of eigenvectors for all $\sigma$ is the columns of the Hadamard matrix, and thus $D$ must have the columns of $H_d$ as its eigenvectors.

Our analysis of eigenvalues in Lemma B.1 is closely related to the Fourier analysis of the Boolean hypercube, which has been studied for decades in computer science theory [O'D14]. Fourier analysis of the Boolean hypercube can be seen as an instance of our technique, by setting $a$ to be the all ones vector in $d$ dimensions. It is important in some of our proofs that $a$ is not the all ones vector: kernel matrices from a hyperrectangle with varying side lengths have eigenvectors that approximate finite differences. This is key for our proofs of Theorem 4.4 and 3.4. See Section E.2 for details.

We next give an overview of how we use Lemma B.1 to derive our three applications. We focus on explaining how the matrices described by Lemma B.1 and their eigenvalues arise in each setting.

### B.2 Polynomial Method Converse

#### B.2.1 Exact Low Rank

We begin by explaining our techniques for the exact low rank case (Theorem 2.5). This theorem seems hard to prove for a number of reasons.

First, we assume very little structure on $f$: in particular, we do not asssume $f$ is differentiable or even continuous. Rather, we assume a much weaker condition than continuity: indeed, Theorem 2.5 holds for all piecewise continuous functions $f$. This is a large space of functions, and in particular covers all non-differentiable continuous functions including oddities such as the everywhere-continuous and nowhere-differentiable Weirstrauss function.

Second, to prove a matrix $M^f$ is low rank, one might typically compute either the determinant or an eigenvalue and show they must be $0$. However, explicit formulas for determinants and eigenvalues can often be complicated, large algebraic expressions in terms of $f$ and elements of $M$. We overcome this barrier by selecting a special family of matrices $M$ whose eigenvalues can be expressed as a simple sum. This family of matrices is restrictive enough to have a set of common eigenvectors (and thus easily computable eigenvalues), but expansive enough to express all finite differences of $f$ in terms of these eigenvalues. Our proof proceeds as follows.

**Step 1:** We begin by showing that if $f : \mathbb{R} \to \mathbb{R}$ preserves low-rank $n \times n$ matrices, and does not have any essential discontunuities of the first kind, then $f$ must be continuous. We do this by constructing a family of $n \times n$ matrices of rank *at most 5* such that, for any jump, point, removeable, or essential-of-second-kind discontinuity that the function $f$ has, one can pick a corresponding matrix from our family that $f$ maps to a full-rank matrix. In other words, such functions $f$ are very far from preserving $n \times n$ low-rank matrices.

**Step 2:** We next show that if a continuous function $f$ preserves low-rank $n \times n$ matrices, then it must be a piecewise polynomial function. We will do this by considering the family of $n \times n$ kernel matrices of a hyperrectangle, from Section B.1. If $f$ preserves low-rank matrices, then it must, in particular, map all these matrices to matrices which do not have full rank.

For any fixed $i$, we will show that the $d$-th order finite difference of $f$ at point $x$ can be written as a linear combination of the $i$-th eigenvalue $\lambda_i$, of a number of different matrices in our family (see Lemma B.1 for definition of $\lambda_i$ and our kernel matrices). Hence, if $\lambda_i$ is $0$ for all of the above-mentioned kernel matrices, this will imply that the $d$-th order finite difference of $f$ is $0$ at every point $x$, and thus $f$ is a polynomial of degree at most $d$.

Although we are guaranteed that one of the eigenvalues of each kernel matrix is $0$, we are not guaranteed that there is a fixed $i$ such that it is always the $i$-th eigenvalue which is $0$.

In order to address this, we apply the Baire Category Theorem from topology to the zero-sets of $\lambda_i$ for each fixed $i$. Roughly, this theorem allows us to show that for all $x \in \mathbb{R}$ (except for a set whose intersection with any finite interval is finite), one can manipulate which matrices determine the finite difference of $f$ at $x$ to ensure that they all have the same eigenvalue $\lambda_i$ equal to $0$. Working through the details, this implies that $f$ is a piecewise polynomial.

**Step 3:** Next, we show that the function $f$ must be exactly a polynomial. From step 2, we know that $f$ is piecewise polynomial. We then use a series of algebraic manipulations, and the fact that a linear combination of $\lambda_i$ from different matrices gives the finite difference of $f$, to show that each $d$-th finite difference of $f$ evaluated at any point for any gap is $0$. This implies that $f$ is a polynomial, finishing our proof.

#### B.2.2 Approximate Low Rank

Our proof of Theorem 2.6 is similar to the exact setting (Theorem 2.5). The main new technical difficulty which arises is that we must more carefully bound the finite differences in terms of the eigenvalues; in the previous proof, we could assume one of the eigenvalues is $0$ and so many terms cancelled out. We omit further details here in the interest of space, but refer the reader to Section D for more details.

## B.3 Metric Transforms

At the onset, Theorem 4.4 seemed difficult to prove for a number of reasons.

It is known, and not hard to show, that any Bernstein function transforms squared Euclidean distances to squared Euclidean distances [BCR84]. It was also known that Manhattan distances isometrically embed into squared Euclidean distances. Thus Theorem 4.4 immediately implies that Bernstein functions are equivalent to functions that transform squared Euclidean to squared Euclidean distances. This equivalence is Schoenberg's foundational theorem on Euclidean metric transforms [Sch37], which is considered difficult to prove from scratch.

Schoenberg's proof uses multivariable calculus and complex analysis in Hilbert space, which has a natural Euclidean distance structure; Manhattan distances do not have such structure, so we cannot proceed in this way to prove Theorem 4.4. Our proof takes an entirely new approach, and our result is stronger than Schoenberg's.

Moreover, we note that most squared Euclidean distances are not Manhattan distances: indeed, most squared Euclidean distances are not even metrics. Thus, it was highly conceivable before our work that non-Bernstein functions could transform Manhattan distances to squared Euclidean distances. Our work rules this possibility out. It was also unclear prior to our work why functions transforming Manhattan to Manhattan, and functions transforming Manhattan to squared Euclidean, should be the same set of functions.

Notably, we do not assume any kind of structure on metric transforms $f$: we do not assume $f$ is bounded, continuous, Fourier-transformable, and so forth. Thus, our theorem applies to any conceivable function with no underlying structure assumed at all.

**Step 1:** First, we show that a function transforms Manhattan distances to squared Euclidean distances, if and only if $f$ applied entrywise to our kernel matrices from hyperrectangles (see Lemma B.1 for how these are defined) always results in a matrix whose eigenvalues are all negative except for $\lambda_1$. Here $\lambda_i$ is defined as in Lemma B.1. This follows from the well-known fact that all Manhattan distances can be isometrically embedded into Manhattan distances between points on a hyperrectangle, combined with a criterion of Schoenberg [Sch35] on when a given set of $\binom{n}{2}$ distances can be realized as pairwise squared Euclidean distances from $n$ points.

**Step 2:** Next, we show that only Bernstein functions transform Manhattan distances to squared Euclidean distances. We do this by showing the $d$-th order finite difference of $f$ evaluated at $x$, can be written as $(-1)^d$ times the limit of a sequence of eigenvalues of well-chosen kernel matrices from hyperrectangles. We can bound this limit using step 1, to show that if $f$ transforms Manhattan distances to squared Euclidean distances, then the $d$-th order finite difference for $f$ have the opposite sign as $(-1)^d$. This property implies that $f$ is Bernstein, even without assuming a priori that $f$ is bounded or continuous.

**Step 3:** We will then show that any function that transforms Manhattan distance to squared Euclidean distance, must transform Manhattan distances to Manhattan distances.

We first show that $f$ transforms Manhattan distances to squared Euclidean distances if and only if it transforms Manhattan distances from hyperrectangle corners to squared Euclidean distances. We then find the explicit embedding of these squared Euclidean distances via a classic idea of Schoenberg [Sch35], which will reveal that these transformed distances can be embedded as squared Euclidean distances from corners on a different, higher dimensional hyperrectangle. Squared Euclidean distances from corners of a hyperrectangle are isometric to Manhattan distances, by the Pythagorean theorem.

## B.4 Kernel Methods

Theorem 3.4 represents a non-trivial advance in kernel theory for the following reason:

One of the fundamental results of kernel methods is a classification of all Euclidean kernels. From this, one can deduce that a function is a squared Euclidean kernel if and only if it is a completely monotone function. Since Manhattan distances are a measure-zero set of squared Euclidean distances,

it is clear that completely monotone functions are Manhattan kernels, but it is not at all clear that all Manhattan kernels should be completely monotone.

The proof steps for kernel methods are very similar to those for metric transforms. The reason is that there is a known connection between matrices of squared Euclidean distances between pairs of points, and matrices of inner products between pairs of points.

The main difference between the proof of Theorem 3.4 on kernels and the proof of Theorem 4.4 on metric transforms (whose steps are listed in Section B.3) is that in Step 1 of our proof on kernels, we show that a function is a Manhattan distance kernel if and only if $f$ applied entrywise to our kernel matrices from hyperrectangles always results in a matrix whose eigenvalues are all non-negative. We propagate this change through the proof steps accordingly.

# C  Polynomial Method Converse

The major goal of this section is to prove Theorem C.11, the formal restatement of Theorem 2.5. This section is organized as follows

- In Section C.1, we restate some preliminaries about kernel matrices from real hyperrectangles. We define matrices $M(a)$, $M^f(a)$, and eigenvalues $\lambda_i^f(a)$.

- In Section C.2, we prove that low degree polynomials are the only functions such that $M^f(a)$ has an eigenvalue that is the zero function in terms of $a \in \mathbb{R}^d$.

- In Section C.3, we show that for any function that preserves low rank, one eigenvalue of $M^f(a)$ must be zero in terms of $a$.

- In Section C.4, we provide an algebraic computation, which is a key step for equating a sum of eigenvalues with the formula for finite differences.

- In Section C.5, we formally relate a key sum of $f$ evaluated at various points, with the finite differences of $f$.

- In Section C.6, we prove that only low-degree polynomials preserve low rank. This proves the main result of this section, Theorem C.11, which is a formal restatement of Theorem 2.5.

- In Section C.7, we show a large class of discontinuous functions do not preserve low-rank matrices.

## C.1  Preliminaries

We start by defining the matrix $M(a)$ from the real hyperrectangle, and its eigenvalues.

**Definition C.1** (Matrix $M(a)$). *Consider a mapping $B : \{0, 1, \ldots 2^d - 1\} \to \{0, 1\}^d$ corresponding to the conversion of integers into $d$-digit binary strings, which we interpret as $d$ dimensional $0-1$ vectors. For any fixed vector $a \in \mathbb{R}^d$, we define matrix $M(a)$*

$$M(a)_{i,j} := \langle a, B(|i - j|) \rangle.$$

**Definition C.2** (Eigenvalues of $M(a)$). *For each matrix $M(a) \in \mathbb{R}^{n \times n}$, we established previously that $f(M(a)) \in \mathbb{R}^{n \times n}$ has eigenvectors equal to the Hadamard matrix columns, and the corresponding eigenvalues are:*

$$\lambda_i^f(a) = \sum_{b \in \{0,1\}^d} (-1)^{\langle B(i), b \rangle} \cdot f(\langle b, a \rangle)$$

We also give the following definition:

**Definition C.3** (Real hyperrectangle). *The $d$-dimensional real hyperrectangle parameterized by $d$ variables $a_1, \ldots a_d > 0$ is the convex hull of the $2^d$ points $\{\pm a_1/2, \ldots \pm a_d/2\}$.*

## C.2  Functions with Algebraically Zero Eigenvalues

The goal of this section is prove Lemma C.4.

**Lemma C.4** (Only polynomials have a zero eigenvalue). *For any function $f$, any $n$ that is a power of 2, and $d := \log n + 1$: we can find $M : \mathbb{R}^d \to \mathbb{R}^{n \times n}$ and $\lambda_i^f : \mathbb{R}^d \to \mathbb{R}$ satisfying:*

1. *$M(a)$ has rank $\leq d$ for all $a \in \mathbb{R}^d$*

2. *$\lambda_1^f(a) \dots \lambda_n^f(a)$ is the full set of eigenvalues of $f(M(a))$, for all $a \in \mathbb{R}^d$.*

3. *If there exists $i \in [n]$ such that $\lambda_i^f(a) = 0$ for all $a \in \mathbb{R}^d$, then $f$ is a degree $d \leq \log n + 1$ polynomial.*

*Proof.* We note that for any $\varepsilon > 0$,

$$(-1)^{\langle B(i), 1 \rangle} \sum_{b \in \{0,1\}^d} (-1)^{\|b\|_1} \cdot \lambda_i^f(a + \epsilon b) = \sum_{b \in \{0,1\}^d} (-1)^{\|b\|_1} \cdot f(\langle a + \epsilon b, 1 \rangle) \quad (1)$$

by the following computation:

$$(-1)^{\langle B(i), 1 \rangle} \sum_{b \in \{0,1\}^d} (-1)^{\|b\|_1} \cdot \lambda_i^f(a + \epsilon b)$$

$$= (-1)^{\langle B(i), 1 \rangle} \sum_{b_1 \in \{0,1\}^d} (-1)^{\|b_1\|_1} \cdot \left( \sum_{b_2 \in \{0,1\}^d} (-1)^{\langle B(i), b_2 \rangle} f(\langle b_2, a + \epsilon b_1 \rangle) \right)$$

$$= (-1)^{2 \langle B(i), 1 \rangle} \sum_{b \in \{0,1\}^d} (-1)^{\|b\|_1} \cdot f(\langle a + \epsilon b, 1 \rangle)$$

$$= \sum_{b \in \{0,1\}^d} (-1)^{\|b\|_1} \cdot f(\langle a + \epsilon b, 1 \rangle)$$

where the first equality follows from the definition of $\lambda_i^f$ and the second equality follows from Lemma C.9.

It follows that if $\lambda_i^f = 0$, then

$$\sum_{b \in \{0,1\}^d} (-1)^{\|b\|_1} \cdot f(\langle a + \epsilon b, 1 \rangle) = 0$$

for all $\epsilon$ and $a$. By Lemma C.10, the above equation implies that $\Delta_\epsilon^d[f](\langle a, 1 \rangle) = 0$ for all $\epsilon$. If all $d^{th}$ order finite differences are equal to 0 at $a$, then the $d^{th}$ derivative of $f$ at $a$ exists and is also equal to 0.

Therefore, $f$ is at most a degree $d$ polynomial as desired. Thus, we complete the proof.

$\square$

We conjecture that the above lemma is true without requiring $n$ to be a power of 2, but it is not necessary for our other results.

## C.3 Eigenvalues of Low Rank Preserving Functions

The goal of this section is to prove Lemma C.5.

**Lemma C.5** (One Eigenvalue is identically zero). *If $n$ is a power of 2 and given:*

1. *A function $f : \mathbb{R} \to \mathbb{R}$ with no essential discontinuities of the first type*

2. *A function $M : \mathbb{R}^d \to \mathbb{R}^{n \times n}$, mapping $d := \lceil \log n \rceil$ dimensional vectors to $n$ dimensional matrices.*

3. *A set of $n$ functions $\lambda_1^f, \lambda_2^f, \cdots, \lambda_n^f$ such that each $\lambda_i^f : \mathbb{R}^d \to \mathbb{R}$, and $\lambda_1^f(a) \dots \lambda_n^f(a)$ is the full set of eigenvalues of $f$ applied entry-wise to $M(a)$ for all $a \in \mathbb{R}^d$,*

*Then if $f$ transforms matrices $M(a)$ to rank $< n$ for all $a \in \mathbb{R}^d$, then there exists $i \in [n]$ where function $\lambda_i^f = 0$.*

*Proof.* Lemma C.8 says that if $f$ preserves low rank and has no essential discontinuities of the first type, then it is piecewise polynomial. We will prove that if it preserves low rank and is piecewise polynomial, it must be polynomial.

We consider the family of matrices $(J + M(a))^f$, where $J \in \mathbb{R}^{n \times n}$ is the all ones matrix. Note that $M(a)$ has rank at most $d + 1$ and one of the eigenvectors is all ones vector. Thus, matrices of the form $J + M(a)$ have rank at most $d + 1$.

Suppose otherwise, that $f$ preserves low rank and is piecewise polynomial but not polynomial. Without loss of generality, we can assume that $f(x) = P_1(x)$ on $[r, s]$ and $f(x) = P_2(x)$ on $[s, t]$ for some $r < s < t$, for some polynomials $P_1 \neq P_2$.

There exists an open set $X \subset \mathbb{R}$ and an open set $Y \subset \mathbb{R}^d$ such that $x \in [r, s]$ and $x + \langle y, a \rangle$ is in $[s, t]$ for all $x \in X, y \in Y$, and non-zero $a \in \{0, 1\}^d$. $X, Y$ can be attained by choosing $X$ to be the set of $x$ satisfying $s - \epsilon < x < s$, and and $Y$ to be the set of $y$ with $\epsilon < y_j < 2\epsilon$, for any $0 < \epsilon < (t - s)/(2d)$, for all $1 \leq j \leq d$.

We use our eigenvalue computation in Definition C.2 to see that:

$$\lambda_i^f(xJ + M(y)) = P_1(x) + \sum_{a \in \{0,1\}^d \setminus \mathbf{0}_d} (-1)^{\langle B(i), a \rangle} P_2(x + \langle y, a \rangle). \tag{2}$$

Here, $B(i)$ is the $d$ dimensional binary representation of $i \in [n]$ in Eq. (2).

If $f$ preserves low rank, then for each $x \in X$ and for each $y \in Y$, there exists an $i$ such that the RHS of Eq. (2) identically zero. Since $X \times Y$ is an open set in $\mathbb{R}^{d+1}$, there exists $i$ such that the roots of the RHS of Eq. 2 has non-zero measure. Since this RHS is a multivariate polynomial in $x$ and $y_k$ for $1 \leq k \leq d$, and since the only multivariate polynomial with non-zero measure is 0 everywhere, then $\lambda_i^f(xJ + M(y))$ must be 0 for all $x \in \mathbb{R}^d$ and $y \in \mathbb{R}^d$ for our chosen value of $i$, and thus for this value of $i$, $\lambda_i^f(M(a)) = 0$ for all $a$.

$\square$

We conjecture that Lemma C.5 is true for all $n$, not just powers of 2. However, that is not necessary for our other results.

The proof above holds assuming Lemma C.8. We now build up a series of Lemmas leading up to that point. First, we need a technical lemma.

**Lemma C.6** (Locally Zero Eigenvalue Implies Vanishing $d^{th}$ Derivative). *Let $n$ be a power of 2 and $d = \lceil \log n \rceil + 1$. If there exists $i \in [n]$ such that, for fixed $a \in \mathbb{R}^d$, we have*

$$\lambda_i^f(a + h) = 0$$

*for all $h \in \mathbb{R}^d$ with $\|h\|_\infty < H_a$ for some $H_a > 0$, then $\Delta_h^d[f](\langle a, 1 \rangle) = 0$ for all $\|h\|_\infty < H_a$ and thus $f^{(d)}(\langle a, 1 \rangle)$ exists and is equal to 0.*

*Proof.* For fixed $a$, $d$, and $i$, we know by Eq. (1) and Lemma C.10 that $\Delta_h^d[f](a)$ can be written as a linear combination of $\sum_{k \in K} c_k \lambda_i(a + h_k)$ for some $c \in R^{|K|}$ and $h_k < H_a$ for all $k$. Lemma C.6 follows. $\square$

**Lemma C.7.** *Let $T_1, \ldots T_k \subset \mathbb{R}^d$ be closed sets such that*

$$\cup_{i \in [k]} T_i = \mathbb{R}^d.$$

*Let $R_i$ be the interior of $T_i$. Then when taking the union of the projections of $R_i$ onto the line $c \cdot \mathbf{1}_d \in \mathbb{R}^d$ for $c \in \mathbb{R}$, the result is the entire line except for a set $Q$ of points, where $Q$ intersected with any finite interval contains only finitely many points.*

*Proof.* First, we prove that this holds when $d = 1$ and $T_i$ are the closure of open sets. We will reduce the general case to the case in the previous sentence.

**One-dimensional case**    If $T_i$ is the closure of open sets in one dimension, then it is the union of disjoint closed sets where each finite interval on the real line contains only finitely many closed sets contained in $T_i$. We can assume without loss of generality that the interiors of $T_i$ are disjoint. In this case, the set $T_i \setminus R_i$ is the union (over all $i$) of endpoints of closed intervals in $T_i$. It follows that $T_i \setminus R_i$ is a set $Q$ of points such that every finite interval on the real line contains only finitely many elements of $Q$, as desired.

**Reduction to one-dimensional case.**    We will reduce to the case where each $T_i$ is the closure of its interior. Once we have this, we can project $T_i$ and $R_i$ onto the line $\ell = c \cdot \mathbf{1}_d, c \in \mathbb{R}$) and thus reduce to the one dimensional case.

First, we show that all points $q$ of the form $(c, c, \ldots c) \in \mathbb{R}^d$ that are not in the union of $R_i$, must be on the boundary of some $R_i$. This will prove that we do not lose any generality by considering the case when $T_i$ is the closure of its interior $R_i$.

If not, then there is some open set $S$ containing $q$ that avoids all $R_i$, so now $S$ is covered by one of the $U_i := T_i \setminus R_i$. The $U_i$ are all closed and have empty interior, so their complements $V_i$ are all open and dense since the $U_i$ cover $S$.

Since the $U_i$ cover $S$, the intersection of the $V_i$ must be empty. However this contradicts the Baire Category theorem (Theorem A.11), which states that the intersection of a family of open and dense sets is also open and dense (and in particular nonempty). This means that $q$ must be on the boundary of some $R_i$, and thus we have proven that we can reduce to the case when $T_i$ is the closure of its interior $R_i$.

**Reduction to the one dimensional case.**    Since we only need to concern with the case where $T_i$ is the closure of its interior $R_i$, this means each point in $T_i \setminus R_i$ is the limit of a sequence of points in $R_i$ for each $i$. Therefore, the projection of each point in $T_i \setminus R_i$ is the limit of a sequence of points in the projection of $R_i$, and thus the projection of $T_i$ onto the line $(c, c, \ldots c) \in \mathbb{R}^d$ is contained in the closure of the projection of $R_i$. This reduces the problem to the one dimensional case where each $T_i$ is the closure of its interior.

$\square$

**Lemma C.8** (Piecewise Polynomial). *If $f$ preserves low rank for all $n \times n$ matrices where $n$ is any power of 2, and has no essential discontinuities of the first type, then $f$ must be piecewise polynomial.*

Since $f$ preserves low rank, we know that for any $a$, there exists an $i \in [n]$ such that $\lambda_i^f(M(a)) = 0$. By Lemma C.13, we know that $f$ must be continuous. Therefore, the $T_i$ of $a$ where $\lambda_i^f(M(a)) = 0$ must satisfy

$$\cup_{i \in [n]} S_i = \mathbb{R}^d.$$

Because $f$ is continuous, $\lambda_i^f$ is continuous for all $i$, and so each $T_i$ is closed. Let $R_i$ be the interior of each $T_i$. We know by Lemma C.6 that for each point $a'$ in $R_i$, we know that $f^{(d)}(\langle a, 1 \rangle) = 0$. By Lemma C.7, this implies $f^{(d)}(x) = 0$ for all $x \in \mathbb{R} \setminus Q$, where $Q \subset \mathbb{R}$ has the property that its intersection with any finite interval is finite. This implies that $f$ is a piecewise polynomial.

## C.4 Bridging Eigenvalues and Finite Differences

The goal of this section is to prove Lemma C.9, a key lemma which is used to relate eigenvalues and finite differences.

**Lemma C.9** (Rewriting the sum).

$$\sum_{b_1 \in \{0,1\}^d} (-1)^{\|b_1\|_1} \left( \sum_{b_2 \in \{0,1\}^d} (-1)^{\langle B(i), b_2 \rangle} f(\langle b_2, a + \epsilon b_1 \rangle) \right)$$
$$= (-1)^{\langle B(i), \mathbf{1} \rangle} \sum_{b \in \{0,1\}^d} (-1)^{\|b\|_1} \cdot f(\langle a + \epsilon b, \mathbf{1} \rangle)$$

*where $a$ and $b$ are $d$-dimensional vectors.*

*Proof.* First, we can show: If $b_2$ is a $d$ dimensional vector with any 0s in its vector notation, we know

$$\sum_{b_1 \in \{0,1\}^d} (-1)^{\|b_1\|_1} f(\langle b_2, a + \epsilon b_1 \rangle) = 0 \tag{3}$$

for any $\epsilon$, and any constant $d$ dimensional vector $a$. The reason is if $b_2$ has any 0's in its vector notation, then flipping the corresponding bit in $b_1$ causes $(-1)^{\|b_1\|_1}$ to change sign, while leaving $\langle b_2, a + \epsilon b_1 \rangle$ unchanged.

Now, we know that:

$$\sum_{b_1 \in \{0,1\}^d} (-1)^{\|b_1\|_1} \left( \sum_{b_2 \in \{0,1\}^d} (-1)^{\langle B(i), b_2 \rangle} f(\langle b_2, a + \epsilon b_1 \rangle) \right)$$

$$= \sum_{b_2 \in \{0,1\}^d} (-1)^{\langle B(i), b_2 \rangle} \left( \sum_{b_1 \in \{0,1\}^d} (-1)^{\|b_1\|_1} f(\langle b_2, a + \epsilon b_1 \rangle) \right)$$

$$= (-1)^{\langle B(i), \mathbf{1} \rangle} \left( \sum_{b_1 \in \{0,1\}^d} (-1)^{\|b_1\|_1} f(\langle \mathbf{1}, a + \epsilon b_1 \rangle) \right).$$

where the first equality follows by rearranging sums, and the second equality follows from Eq. (3). This completes the proof.

$\square$

## C.5 Function Sums and Finite Differences

The goal of this section is to prove Lemma C.10.

**Lemma C.10** (Function Sums and Finite Differences). **Part 1.** *Suppose the $d^{th}$ derivative of $f$, denoted as $f^{(d)}$, is continuous. Then, we have*

$$\lim_{\epsilon \to 0} \epsilon^{-d} \sum_{b \in \{0,1\}^d} (-1)^{\|b\|_1} \cdot f(\langle a + \epsilon b, \mathbf{1} \rangle) = f^{(d)}(\langle a, \mathbf{1} \rangle).$$

**Part 2.** *Suppose $\Delta_\epsilon^d[f](z)$ is the d-th finite difference of function $f$, then we have*

$$\sum_{b \in \{0,1\}^d} (-1)^{\|b\|_1} \cdot f(\langle a + \epsilon b, \mathbf{1} \rangle) = \Delta_\epsilon^d[f](\langle a, \mathbf{1} \rangle).$$

*Proof.* We have:

$$\sum_{b \in \{0,1\}^d} (-1)^{\|b\|_1} \cdot f(\langle a + \epsilon b, \mathbf{1} \rangle) = \sum_{s=0}^{d} (-1)^s \binom{d}{s} \cdot f(\langle a + \epsilon b, \mathbf{1} \rangle)$$

$$= \sum_{s=0}^{d} (-1)^s \binom{d}{s} \cdot f(\langle a, \mathbf{1} \rangle + s\epsilon)$$

$$= \int_{[0,\epsilon]^d} f^{(d)}(\langle a + x, \mathbf{1} \rangle) \mathrm{d}x \tag{4}$$

$$= \Delta_\epsilon^d[f](\langle a, \mathbf{1} \rangle)$$

where the first and second equality follow from grouping $b$ by the number of ones it has, which we denote as $s$, the third step follows from the fundamental theorem of calculus, and the last step follows from definition of finite difference.

In addition, we note that the above calculation is independent of $i$.

Thus:

$$\lim_{\epsilon \to 0} \epsilon^{-d} \sum_{b \in \{0,1\}^d} (-1)^{\|b\|_1} \cdot f(\langle a + \epsilon b, \mathbf{1} \rangle)$$

$$= \lim_{\epsilon \to 0} \epsilon^{-d} \int_0^\epsilon \int_0^\epsilon \dots \int_0^\epsilon f^{(d)}(\langle a + x, \mathbf{1} \rangle) \mathrm{d}x_1 \dots \mathrm{d}x_d$$

$$= f^{(d)}(\langle a, \mathbf{1} \rangle)$$

where the first equality follows from Eq. (4) and the last equality follows from the continuity of $f^{(d)}$. This completes the proof of Lemma C.10. $\qquad\square$

## C.6 No Functions Other Than Polynomials Preserve Low Rank

In this section, we prove main result Theorem C.11 using Lemma C.4 and Lemma C.5.

**Theorem C.11** (Formal statement of Theorem 2.5). *Suppose the function $f : \mathbb{R} \to \mathbb{R}$ does not have any essential discontinuities of the first kind. For any positive integer $n \geq 2$, the function $f$ preserves low rank matrices if and only if $f$ is a polynomial of degree less than $\lceil \log_2(n) \rceil$.*

*Proof.* First, we prove it for $n$ as powers of 2, and then we generalize to all $n$.

For now, suppose $n$ is a power of 2. Suppose $f$ is a function without essential discontinuities of the first type. By Lemma C.4, we can find $M : \mathbb{R}^d \to \mathbb{R}^{n \times n}$ and $\lambda_i^f : \mathbb{R}^{\log n + 1} \to \mathbb{R}$ such that the image of $M$ has rank $\leq \log n + 1$, and $\{\lambda_i^f(a)\}_{i \in [n]}$ is the full set of eigenvalues of $M(a)$. Further, if there exists $i \in [n]$ with function $\lambda_i^f(a) = 0$ for all $a$, then $f$ is a degree $d \leq \log n + 1$ polynomial.

Now, suppose that $f$ is a function that transforms all rank $\log n + 1$ matrices to rank $< n$ matrices. Then it must transform all matrices $M(a)$ to rank $< n$ matrices. By Lemma C.5, it must follow that $\lambda_i^f = 0$ for some $i$. However, we just established via Lemma C.4 that if $\lambda_i^f = 0$, then $f$ is a degree $d \leq \log n + 1$ polynomial. This completes the proof of Theorem C.11 if $n$ is a power of 2.

The statement for all $n$ follows directly from the statement for $n$ a power of 2. This is because of the fact from linear algebra that if an $N \times N$ matrix $M$ has full rank, then for any fixed $n \leq N$, there exists an $n \times n$ minor of $M$ with full rank. (This follows, for instance, from expansion by minors.) Suppose $n$ is not a power of 2, and let $2^d$ be the smallest power of 2 bigger than $n$. If $f$ is not a low-degree polynomial, our work as-stated proves that there exists a low rank $2^d \times 2^d$ matrix $M$ where $M^f$ is full rank. By the previous, there exists an $n \times n$ minor $M_n$ of $M$ (with low rank, since its rank is less than that of $M$) where $M_n^f$ is full rank. Therefore, $f$ cannot preserve low rank matrices of dimension $n \times n$, if $f$ is not a low degree polynomial.

$\qquad\square$

## C.7 Proof of Continuity

The goal of this section is to prove the following lemma, which shows that a large class of functions $f : \mathbb{R} \to \mathbb{R}$ do not preserve low-rank matrices.

**Lemma C.12.** *Suppose $f : \mathbb{R} \to \mathbb{R}$ has any point $c \in \mathbb{R}$ such that $\lim_{x \to c^+} f(x)$ exists and $f(c) \neq \lim_{x \to c^+} f(x)$. Then, $f$ does not preserve low-rank matrices. The same is true with '$c^+$' replaced by '$c^-$'.*

Lemma C.12 will follow from Lemma C.15, which we now build up to and prove.

**Lemma C.13.** *Suppose that $f : \mathbb{R} \to \mathbb{R}$ satisfies: $\lim_{x \to 0} f(x) \neq f(0)$. Then, for any positive integer $n$, there is a rank-2 matrix $M \in \mathbb{R}^{n \times n}$ such that $M^f$ has full rank.*

*Proof.* Let $a = f(0)$ and $b = \lim_{x \to 0} f(x)$. By assumption, $a \neq b$. We define the $n \times n$ matrix $A$ as follows

$$A_{i,j} = \begin{cases} a & \text{if } i = j; \\ b & \text{otherwise.} \end{cases}$$

The definition of $b$ means that, for all $\epsilon > 0$, there is a $\delta > 0$ such that if $x \in \mathbb{R}$ satisfies $0 < |x| < \delta$, then $|f(x) - b| < \epsilon$. Let us pick $\varepsilon > 0$ to be sufficiently small so that

$$0 < \epsilon \leq \frac{0.1}{n \cdot n! \cdot \max\{|a|, |b|, |b+\epsilon|, |b-\epsilon|\}^{n-1}} \cdot \det(A),$$

and let $\delta > 0$ be the corresponding value.

Let $M_{i,j} = \frac{\delta}{n} \cdot (i - j)$. We can see that $\operatorname{rank}(M) = 2$.

From assumption, we have

$$M_{i,j}^f = \begin{cases} a & \text{if } i = j; \\ \in [b - \epsilon, b + \epsilon] & \text{otherwise.} \end{cases}$$

We can upper bound

$$
\begin{aligned}
|\det(M^f) - \det(A)| &= \left| \sum_{\sigma \in S_n} \operatorname{sgn}(\sigma) \prod_{i=1}^n M_{i,\sigma_i}^f - \sum_{\sigma \in S_n} \operatorname{sgn}(\sigma) \prod_{i=1}^n A_{i,\sigma_i} \right| \\
&\leq \sum_{\sigma \in S_n} \left| \prod_{i=1}^n M_{i,\sigma_i}^f - \prod_{i=1}^n A_{i,\sigma_i} \right| \\
&\leq \sum_{\sigma \in S_n} \epsilon \cdot n \cdot (\max\{|a|, |b|, |b - \epsilon|, |b + \epsilon|\})^{n-1} \\
&\leq n! \cdot \epsilon \cdot n \cdot (\max\{|a|, |b|, |b - \epsilon|, |b + \epsilon|\})^{n-1}
\end{aligned}
$$

where the first step follows from definition of matrix determinant.

Then we can bound

$$
\begin{aligned}
\det(M^f) &= \det(A) + \det(M^f) - \det(A) \\
&\geq \det(A) - |\det(M^f) - \det(A)| \\
&\geq \det(A) - 0.1 \det(A) \\
&= 0.9 \det(A) \\
&> 0.
\end{aligned}
$$

$\square$

By picking a matrix $M$ whose entries are all nonnegative, we can extend the same idea to functions $f$ where only one side of $\lim_{x \to 0} f(x)$ must exist:

**Lemma C.14.** *Suppose that $f : \mathbb{R} \to \mathbb{R}$ satisfies: $\lim_{x \to 0^+} f(x) \neq f(0)$. Then, for any positive integer $n$, there is a rank-4 matrix $M \in \mathbb{R}^{n \times n}$ such that $M^f$ has full rank.*

*Proof.* The matrix we use is $M_{i,j} = \frac{\delta}{n} \cdot (i - j)^2$, which has $\operatorname{rank}(M) = 4$. We then proceed exactly as in Lemma C.13. $\square$

**Lemma C.15.** *Suppose that $f : \mathbb{R} \to \mathbb{R}$ and $c \in \mathbb{R}$ satisfy: $\lim_{x \to c^+} f(x) \neq f(c)$. Then, for any positive integer $n$, there is a matrix $M \in \mathbb{R}^{n \times n}$ of rank at most 5 such that $M^f$ has full rank.*

*Proof.* The matrix we use is $M_{i,j} = \frac{\delta}{n} \cdot (i - j)^2 + c$, which has $\operatorname{rank}(M) \leq 5$. Again, proceed as in Lemma C.13. $\square$

## C.8 Comparison with Prior Work

The work of [GKR17] classifies what functions transform low rank matrices into low rank matrices. They show:

**Theorem C.16** (Theorem B in [GKR17]). *For integers $n \geq 2, 1 \leq k < n - 1$, and $2 \leq l \leq n$, suppose $f$ has $k$ derivatives. Then the following are equivalent:*

1. *f transforms rank $l$ PSD $n$ by $n$ matrices into matrices whose rank is upper bounded by $k$.*

2. *f is a polynomial $\sum_{t=1}^{r} a_t x^{i_t}$ for some $a_t \in \mathbb{R}$ and some $i_t \in \mathbb{N}$ such that*

$$\sum_{t=1}^{r} \binom{i_t + l - 1}{l - 1} \leq k$$

*Moreover, if $k \leq n - 3$, we don't need the assumption that $f$ has $k$ derivatives.*

This is a near-full classification of functions that transform low rank matrices into low rank matrices, with a few holes: first, the primary piece of the theorem requires $k$-fold differentiability of $f$, something that doesn't hold true for commonly used functions such as the ReLU function. The part of the theorem which doesn't require $k$-fold differentiability forces $k \leq n - 3$. As discussed in [GKR17], these existing techniques are fundamentally incapable of extending to the settings when to $k = n - 2, n - 1$.

In contrast to Theorem C.16, our Theorem C.11 uses very different techniques, and addresses the case where $k = n - 1$ when $f$ is not required to be differentiable, which is not covered by Theorem C.16. The consequence of extending to $k = n - 1$ is that we have less control over $l$; our results primarily hold for $l = \lg n + 1$ whereas Theorem C.16 applies for more general $l$. Additionally, our result is not exact: we show that if $f$ preserves low rank, then $f$ must be a polynomial, but we only bound the degree of this polynomial up to a factor of 2, whereas theorem C.16 has tighter control over what kind of polynomial $f$ is. Thus, Theorem C.16 is tighter, but their methods are fundamentally unable to handle the case when $k = n - 2$ or $k = n - 1$ without assuming $k$-fold differentiability. Our result is able to handle the $k = n - 1$ case, at the expense of tight bounds on the polynomial degree of $f$ and at the expense of requiring that $l > \lg n + 1$.

## D   Approximate Polynomial Method Converse

In this section, we generalize the proof in Section C from exact to approximate here.

In Section C, we claim that if a function, when applied termwise, transforms a low rank matrix to a low rank matrix, then it must be a low degree polynomial. We will define a function that approximately preserves low rank, and then prove that if a function approximately preserves low rank, then its $d^{th}$ order finite differences must be bounded in two settings: when $f$ is analytic, and when $f$ is Lipschitz.

**Definition D.1.** *Let $f : \mathbb{R} \to \mathbb{R}$ and $\delta > 0$. We say $f$ $\delta$-approximately preserves low rank matrices if, for every matrix $M \in \mathbb{R}^{n \times n}$ with $\mathrm{rank}(M) < \log n$, the matrix $f(M)$ has at least one eigenvalue in $[-\delta/n, \delta/n]$.*

It is not hard to show the following fact,

**Fact D.2.** *For a fixed vector $a \in \mathbb{R}^d$, let matrix $M(a) \in \mathbb{R}^{n \times n}$ be defined as Definition C.1. Let $f : \mathbb{R} \to \mathbb{R}_+$, then we have*

$$\max_{i \in [d]} |\lambda_i(M(a))| = \sum_{b \in \{0,1\}^d} f(\langle a, b \rangle)$$

*Proof.* By definition of eigenvalue of matrix $M(a)$, we have

$$\max_{i \in [d]} |\lambda_i(M(a))| \leq \sum_{b \in \{0,1\}^d} |f(\langle a, b \rangle)| = \sum_{b \in \{0,1\}^d} f(\langle a, b \rangle)$$

where the second step follows from $f$ is a positive function.

On the other hand, we also know there is an eigenvalue is equal to $\sum_{b \in \{0,1\}^d} f(\langle a, b \rangle)$. Thus,

$$\max_{i \in [d]} |\lambda_i(M(a))| \geq \sum_{b \in \{0,1\}^d} f(\langle a, b \rangle).$$

$\square$

## D.1 Main Approximate Result

Let $d \in \mathbb{Z}^+, a \in \mathbb{R}^d, h_a \in \mathbb{R}, n = 2^d$.

**Theorem D.3.** *Let $d = \log n$. Let $\delta \in (0,1)$ denote some sufficiently small parameter. Let function $f$ satisfy:*

$$\frac{\min_{i \in [n]} |\lambda_i^f(M)|}{\max_{i \in [n]} |\lambda_i^f(M)|} \leq \delta/n \tag{5}$$

*for all rank $d + 1$ matrices $M$. Let $K_a = \sum_{b \in \{0,1\}^d} f(\langle a, b \rangle)$.*

**Part 1.** *If $f$ is real analytic, there exists an $H_a > 0$ such that for all $h < H_a$, we have*

$$\Delta_h^d[f](\langle a, \mathbf{1} \rangle) \leq \delta K_a$$

**Part 2.** *If $f$ is an $L$-Lipschitz function, then for all $h \geq 0$, we have*

$$\Delta_h^d[f](\langle a, \mathbf{1} \rangle) \leq \delta K_a + hLdn.$$

*Proof.* We will consider two regimes separately:

- Not assuming $f$ is Lipschitz (Lemma D.4).

- Assuming $f$ is (Lipschitz D.5).

$\square$

## D.2 Real Analytic Functions

**Lemma D.4** (real analytic functions with no Lipschitz assumption, part 1 of Theorem D.3). *We have*

$$\lim_{h \to 0} \Delta_h^d[f](\langle a, 1 \rangle) \leq \delta K_a.$$

*Proof.* Let $h = \epsilon$. By Lemma C.10

$$\sum_{b \in \{0,1\}^d} (-1)^{\|b\|_1} \cdot f(\langle a + \epsilon b, \mathbf{1} \rangle) = \Delta_\epsilon^d[f](\langle a, \mathbf{1} \rangle).$$

By proof of Lemma C.4, we have

$$\sum_{b \in \{0,1\}^d} (-1)^{\|b\|_1} \cdot f(\langle a + \epsilon, \mathbf{1} \rangle) = (-1)^{\langle B(i), 1 \rangle} \sum_{b \in \{0,1\}^d} (-1)^{\|b\|_1} \cdot \lambda_i^f(a + \epsilon b)$$

Let us pick $i^* \in [d]$ to be the index that $\lambda_{i*}^f(a)$ is the smallest eigenvalue for matrix $M(a)$. Choose $H_a$ to be such that $\lambda_{i*}^f(M(a + \varepsilon'b)) \leq \delta_n$ for all $\varepsilon' < H_a$.

Thus, we have

$$\Delta_\epsilon^d[f](\langle a, \mathbf{1} \rangle) = (-1)^{\langle B(i^*), 1 \rangle} \sum_{b \in \{0,1\}^d} (-1)^{\|b\|_1} \cdot \lambda_{i*}^f(a + \epsilon b)$$

$$\leq 2^d \cdot |\lambda_{i*}^f(a)|$$

$$\leq 2^d K_a \delta/n$$

$$= \delta K_a$$

. where the second step follows from the fact that $H_a$ is chosen so that $\lambda_{i*}^f(a + \varepsilon b) \leq \delta/n$ for all $\varepsilon < H_a$. This is doable for all real analytic functions $f$. $\square$

## D.3 Lipschitz Functions

The goal of this section is to prove the following lemma,

**Lemma D.5** ($f$ is $L$-Lipschitz, part 2 of Theorem D.3). *Suppose $f$ is $L$-Lipschitz, we can show*

$$\Delta_h^d[f](\langle a, \mathbf{1} \rangle) \leq \delta K_a + hLdn \tag{6}$$

*Proof.* In the proof, we let $h = \epsilon$. By Lemma C.10, we have

$$\sum_{b \in \{0,1\}^d} (-1)^{\|b\|_1} \cdot f(\langle a + \epsilon b, \mathbf{1} \rangle) = \Delta_\epsilon^d[f](\langle a, \mathbf{1} \rangle).$$

By proof of Lemma C.4, we have

$$\sum_{b \in \{0,1\}^d} (-1)^{\|b\|_1} \cdot f(\langle a + \epsilon, \mathbf{1} \rangle) = (-1)^{\langle B(i), 1 \rangle} \sum_{b \in \{0,1\}^d} (-1)^{\|b\|_1} \cdot \lambda_i^f(a + \epsilon b)$$

Let us pick $i^* \in [d]$ to be the index that $\lambda_{i^*}^f(a)$ is the smallest eigenvalue for matrix $M(a)$.

Thus, we have

$$\begin{aligned}
\Delta_\epsilon^d[f](\langle a, \mathbf{1} \rangle) &= (-1)^{\langle B(i^*), 1 \rangle} \sum_{b \in \{0,1\}^d} (-1)^{\|b\|_1} \cdot \lambda_{i^*}^f(a + \epsilon b) \\
&= (-1)^{\langle B(i^*), 1 \rangle} \sum_{b \in \{0,1\}^d} (-1)^{\|b\|_1} \lambda_{i^*}(a) \\
&\quad + (-1)^{\langle B(i^*), 1 \rangle} \sum_{b \in \{0,1\}^d} (-1)^{\|b\|_1} \cdot (\lambda_{i^*}^f(a + \epsilon b) - \lambda_{i^*}^f(a)) \\
&\leq 2^d |\lambda_{i^*}(a)| + \sum_{b \in \{0,1\}^d} |\lambda_{i^*}^f(a + \epsilon b) - \lambda_{i^*}^f(a)|
\end{aligned}$$

For the first term in the above equation, we can show

$$2^d |\lambda_{i^*}(a)| \leq 2^d \cdot \max_{i \in [d]} |\lambda_i(a)| \delta/n = 2^d \cdot K_a \delta/n = \delta \cdot K_a$$

where the first step follows from Assumption, and the second step follows from Fact D.2 and definition of $K_a$, and the last step follows from $n = 2^d$.

For the second term in the above equation, we can show

$$\begin{aligned}
\sum_{b \in \{0,1\}^d} |\lambda_{i^*}^f(a + \epsilon b) - \lambda_{i^*}^f(a)| &\leq \sum_{b \in \{0,1\}^d} \sum_{c \in \{0,1\}^d} |f(\langle a + \epsilon b, c \rangle) - f(\langle a, c \rangle)| \\
&\leq \sum_{b \in \{0,1\}^d} \sum_{c \in \{0,1\}^d} L \cdot \langle \epsilon b, c \rangle \\
&= L \cdot \epsilon \cdot \langle \sum_{b \in \{0,1\}^d} b, \sum_{c \in \{0,1\}^d} c \rangle \\
&= L \cdot \epsilon \cdot d \cdot 2^d \\
&= L \cdot \epsilon \cdot d \cdot n
\end{aligned}$$

where the second step follows from function $f$ is $L$-Lipschitz.

Thus, we complete the proof. $\qquad\square$

# E  Transforming Manhattan to Euclidean

In this section, we prove Theorem E.2, which states that functions $f$ that transform Manhattan distances to squared Euclidean distances are Bernstein. This section is organized as follows

- In Section E.1, we show that any function $f : \mathbb{R}_{\geq 0} \to \mathbb{R}_{\geq 0}$ transforming Manhattan to squared Euclidean is increasing. This serves as a warm-up for our main theorem.
- In Section E.2, we prove the main result of this section, Theorem E.2. We do this by showing that there is a sequence of $d$-dimensional hyperrectangles such that the kernel matrix associated with them has an eigenvalue which converges to the $d^{th}$ order finite difference of $f$.
- In Section E.3, Lemma E.3 shows $f$ must be bounded and continuous. This lemma is used in the proof of our main result.

## E.1 Manhattan to Euclidean Transforms are Increasing

**Lemma E.1.** *If $f$ transforms Manhattan to squared Euclidean, then $f$ is increasing on $\mathbb{R}_+$.*

*Proof.* We fix $c > 0$ and show $f'(c) \geq 0$. Consider $\chi : [d] \to \{0, 1\}$ which transforms 1 to 1 and everything else to 0. Let $a_1 = \epsilon$ and $a_2, \ldots a_d = \frac{2c}{d}$. Here, $\epsilon$ is a constant which we will adjust later.

The eigenvalue corresponding to $\chi$ (by Lemma I.2) is, by straightforward calculation:

$$\sum_{s=0}^{d-1} \binom{d-1}{s} \left( f\left(\frac{2cs}{d}\right) - f\left(\frac{2cs}{d} + \epsilon\right) \right) \tag{7}$$

If we divide by $2^{d-1}$ and take $d$ to to infinity, the quantity in Eq. (7) becomes

$$f(c) - f(c + \epsilon)$$

for continuous functions $f$. Indeed, nearly all of the probability mass in the binomial coefficients concentrates around $s = d/2$ by the law of large numbers and the limit follows from continuity of $f$ and the boundedness of $f$ on bounded sets established below in Lemma E.3.

Applying Lemma A.8, we see that if $f$ transforms Manhattan to squared Euclidean distances, then $f(c) - f(c + \epsilon) \leq 0$ for any $\varepsilon > 0$. This implies the desired result. $\square$

## E.2 Manhattan to Euclidean Transforms are Bernstein

The goal of this section is to prove Theorem E.2.

**Theorem E.2** (Manhattan to squared Euclidean, formal version of part (1) $\Leftrightarrow$ part (3) of Theorem 4.4)**.** *If $f$ transforms Manhattan distances to squared Euclidean distances, it must be Bernstein.*

*Proof.* Fix a $k$-tuple $\epsilon = (\epsilon_1, \ldots, \epsilon_k)$ of positive real numbers and define

$$\Delta_\epsilon^k(f, t) := f(t) - \sum_{i_1 \in [k]} f(t + \epsilon_{i_1}) + \sum_{i_1 < i_2 \in [k]} f(t + \epsilon_{i_1} + \epsilon_{i+2}) + \ldots + (-1)^k f\left(t + \sum_{i=1}^{k} \epsilon_i\right).$$

Consider $\chi$ that transforms $1, 2, \ldots k$ to 1 and everything else to 0. Let $a_i = \epsilon_i$ for $i \in [k]$ and $a_{k+1} \ldots a_d = \frac{2c}{d}$ where $c$, $k$ and $\epsilon$ are fixed.

The eigenvalue corresponding to $\chi$ is, by direct calculation using Lemma I.2:

$$\lambda_\chi = \sum_{s=0}^{d-k} \binom{d-k}{s} \Delta_\epsilon^k(f, 2sc/d). \tag{8}$$

Eq. (8) is the $d$-dimensional analog of Eq. (7), and this eigenvalue must satisfy $\lambda_\chi \leq 0$ by Lemma A.8. Dividing by $2^{d-k}$ and taking $d$ to infinity, we obtain:

$$\Delta_\epsilon^k(f, c) \leq 0.$$

This is because again the probability mass in the binomial coefficients in Eq. (8) concentrates around the $s = d/2$ coefficient, where we use continuity and boundedness of $f$ for any compact set (guaranteed by Lemma E.3). By Proposition A.2 this implies $f$ is Bernstein (Definition 4.3) since $k, c$ were arbitrary. This completes the proof. $\square$

### E.3 Manhattan to Euclidean Transforms are Bounded

The goal of this section is to prove Lemma E.3.

**Lemma E.3.** *Any function $f : \mathbb{R}_{\geq 0} \to \mathbb{R}_{\geq 0}$ that transforms Manhattan to squared Euclidean is bounded on bounded sets and continuous on $(0, \infty)$.*

*Proof.* By the triangle inequality, $f(x) \leq f(1/2) + f(1/2)$ for all $0 \leq x \leq 1$, so $f$ is bounded on $[0, 1]$. By scaling, from now on we assume $f$ is bounded by 1 on $[0, 1]$.

Now, we show $f$ is continuous on $(0, 1)$. Suppose there is a discontinuity at some point $0 < p < 1$. This means that there exists some $\varepsilon$ such that for all $\delta > 0$, there are $a, b \in [p - \delta, p + \delta]$ such that $f(a) - f(b) \geq \varepsilon$. Since $f(x) \leq 1$ for all $x \in [0, 1]$, this means that for all $\delta < \min\{p, 1 - p\}$, we have that $\frac{f(a)}{f(b)} > 1 + \varepsilon$.

Now, fix some $\varepsilon$ satisfying the above, and some $n = 2k$. Consider points $x_1, \ldots, x_n$ partitioned into sets $A = x_1, \ldots, x_k$ and $B = x_{k+1}, \ldots, x_n$. For some small $\delta$ that we will choose later, pick $a, b \in [p - \delta, p + \delta]$ such that $\frac{f(a)}{f(b)} > 1 + \varepsilon$, and define the metric

$$d(x_i, x_j) := \begin{cases} 0 & i = j \\ a & i, j \in A, i \neq j \\ b & i \text{ or } j \text{ is in } B, i \neq j \end{cases}$$

Now apply $f$: this gives us some metric $d'(x_i, x_j)$ such that

$$d'(x_i, x_j) := \begin{cases} 0 & i = j \\ f(a) & i, j \in A, i \neq j \\ f(b) & i \text{ or } j \text{ is in } B, i \neq j \end{cases}$$

We show that matrix $D'_{i,j} := d'(x_i, x_j)$ is not negative type if $n$ is sufficiently large (as a function of $\varepsilon$). Consider the vector

$$v = (1, 1, \ldots 1, -1, -1, \ldots - 1)$$

with the first $k$ coordinates are ones and the last $k$ coordinates are negative ones. This is orthogonal to the all ones vector, but

$$v^\top D' v = k(k - 1)f(a) - 2k^2 f(b) + k(k - 1)f(b)$$
$$= k(k - 1)f(a) - k(k + 1)f(b).$$

Since $\frac{f(a)}{f(b)} > 1 + \varepsilon$, if we choose $n > 100/\varepsilon$, we will have that

$$k(k - 1) \cdot f(a) - k(k + 1) \cdot f(b) > 0.$$

Therefore, by Lemma A.7, $d'$ does not embed into $\ell_2^2$, Squared Euclidean space.

However, we show that if $\delta$ is sufficiently small (in terms of $n, p$), then $d(x_i, x_j)$ is embeddable into $\ell_1$. First note that the metric $d_1(i, j)$ which equals 0 if $i = j$ and $c$ for some constant $c > 0$ is embeddable into $\ell_1$, by transforming $i$ to $x_i = \frac{c}{2} \cdot e_i$ for all $i$, where $e_i$ is the $i$th unit vector. Likewise, the metric $d_{k,\ell}(i, j)$ which equals 0 if $i = j$ or if $i = k, j = \ell$ or $i = \ell, j = k$ and $c$ otherwise is also embeddable into $\ell_1$, by transforming $i$ to $x_i = \frac{c}{2} \cdot e_i$, except $\ell$ which is sent to $x_\ell = x_k = \frac{c}{2} \cdot e_k$. Now, it is trivial to see that by adding a finite number of these metrics, we still get a metric that is embeddable into $\ell_1$.

But, if $\frac{a}{b} \in \left[1 - \frac{1}{10n^2}, 1 + \frac{1}{10n^2}\right]$, then any metric such that $d(i, j) \in \{a, b\}$ for all $a, b$ can be written as some positive finite combination of $d_1$ and $d_{k,\ell}$ over all $1 \leq k < \ell \leq n$.

Therefore, if $f$ is discontinuous at $p$, we can set $n = \frac{100}{\varepsilon}$, $\delta = \frac{\min(p, 1-p)}{100n^2}$, and the metric on $x_1, \ldots, x_n$ as defined previously. We will have that

$$\frac{a}{b} \in \left[1 - \frac{1}{10n^2}, 1 + \frac{1}{10n^2}\right]$$

whereas $\frac{f(a)}{f(b)} > 1 + \varepsilon$, which means that while $d$ is embeddable into $\ell_1$, $d' = f(d)$ is not embeddable into $\ell_2^2$. Thus, if $f$ is discontinuous at $p$, we have that $f$ cannot transform Manhattan Distances to Squared Euclidean distances.

By scaling the $x$-axis, we have that $f$ is bounded on any interval $[0, a]$ and that $f$ is continuous at all $x > 0$. □

# F  Transforming Manhattan to Manhattan

This section is organized as follows:

- Section F.1 shows how to compute an explicit Euclidean embedding for the distances obtained after applying a Manhattan-to-Euclidean transform to a set of Manhattan distances.

- In Section F.2, we prove Theorem F.3, which states that Manhattan to Manhattan transforms are equivalent to Manhattan to Squared Euclidean transforms. This may be surprising since Squared Euclidean distances are much more general than Manhattan distances.

- Section F.3 gives a discussion for how to generalize our result if the initial metric space was any metric with a vertex-transitive symmetry group, instead of Manhattan distance. We emphasize the importance of vertex transitivity of the symmetry group, and explain why that feature is important for our techniques to apply.

## F.1  Explicit Embeddings for Manhattan to Euclidean Transforms

Suppose $f$ transforms Manhattan distance to squared Euclidean distance. By definition, $f$ satisfies the following: for any $n$ and any $x_1, \ldots x_n \in (\mathbb{R}^{\mathbb{N}}, \ell_1)$, there exist $p_1, \ldots p_n \in (\mathbb{R}^{\mathbb{N}}, \ell_2)$ such that $f(\|x_i - x_j\|_1) = \|p_i - p_j\|_2^2$. We can assume without loss of generality that points $x_1, x_2, \ldots x_n$ are distinct corners of a $d$ dimensional hyperrectangle (Definition C.3), and $n = 2^d$. This is because any point set in $\ell_1$ can be embedded isometrically into $\ell_1$ on corners of a hyperrectangle (Lemma A.3).

**Lemma F.1.** *Let $D$ be the matrix where $D_{i,j} = f(\|x_i - x_j\|_1)$, and let $M := -\frac{1}{2}\Pi D \Pi$. Then $M$ has eigenvectors $H_d$.*

*Proof.* This follows from Lemma B.1 and the definition of $M$. It is critically important that the columns of $H_d$ are orthogonal to the all ones vector (with the exception of the all ones column in $H_d$). □

**Lemma F.2.** *Let $M = H_d \Sigma H_d$ be an eigendecomposition of $M$, where $M$ is defined as in Lemma F.1. If $f$ transforms $\ell_1$ to $\ell_2^2$, then $\Sigma$ has entirely non-negative entries.*

*For each $i$, we use $p_i$ to denote the $i$-th column of $P = \sqrt{\Sigma} H_d$, we have $\langle p_i, p_j \rangle = M_{i,j}$ and $f(\|x_i - x_j\|_1) = \|p_i - p_j\|_2^2$.*

*Proof.* This follows from Lemma A.9 and Lemma F.1. □

## F.2  Manhattan to Manhattan Transforms are Equivalent to Manhattan to Squared-Euclidean Transforms

The goal of this section is to prove Theorem F.3.

**Theorem F.3** (Manhattan to squared Euclidean, formal version of part (2) $\Leftrightarrow$ part (3) of Theorem 4.4)**.** *Any function that transforms Manhattan distances to squared Euclidean distances must transform Manhattan distances to Manhattan distances, and vice versa.*

*Proof.* Let $p_i$ be defined as in Lemma F.2. By construction, the vectors $p_i$ are a subset of the corners of a $2^d$-dimensional hyperrectangle, with side lengths $\sqrt{\Sigma_{i,i}}$. Thus, the pairwise squared Euclidean distances between $p_i$ are isometrically embeddable into $\ell_1$ by Lemma A.4. In other words, $f(\|x_i - x_j\|_1) = \|p_i - p_j\|_2^2 = \|q_i - q_j\|_1$ for some $q_i \in \ell_1$ for all $i, j$. This shows that any $f$ that transforms $\ell_1$ to $\ell_2^2$ transforms $\ell_1$ to $\ell_1$ as desired. □

Note that for any $x_i$, the vectors $q_i$ are finite dimensional and can be explicitly written down in closed form.

### F.3 Metric Transforms for Distances with Group Symmetries

In our proof of Theorem F.3, we exploited that our points $x_1, \ldots x_n$ are points in a hyperrectangle, which has a vertex transitive group symmetry. Similar theories can be generated when the point set lives on any object with a vertex-transitive group symmetry, and the distance measure between points is some function of the Euclidean distance. Such objects include higher dimensional platonic solids, spheres, equilateral triangular prisms, and more.

We remark that the group symmetry must be vertex-transitive to ensure the matrix $D$ in Lemma B.1 has an eigenvector equal to the all ones vector. If this were not the case, Lemma F.1 would no longer hold.

## G Positive Definite Manhattan Kernels

The goal of this section is to prove Theorem G.2, a formal restatement of Theorem 3.4.

The section is organized as follows:

- In Section G.1, we show that positive definite Manhattan kernels map positive numbers to positive numbers. This will be used in our proof of Theorem G.2.
- In Section G.2, we present and prove Theorem G.2. This result classifies all positive definite Manhattan kernels (Definition 3.1), and proves that positive definite Manhattan kernels are equivalent to completely monotone functions.

### G.1 Positive Definite Manhattan Kernels map Positive Reals to Positive Reals

First, we prove the following lemma.

**Lemma G.1.** *If $f$ is a positive definite Manhattan kernel (Definition 3.1), then $f(t) \geq 0$ for all $t \geq 0$.*

*Proof.* Let $\mathcal{X}$ denote metric space $(\mathbb{R}^N, \ell_1)$. For any $N \geq 0$ we consider the points $x_i = \frac{t}{2} e_i \in \mathcal{X}$ for $i \in [N]$ where $e_i = (0, \ldots, 0, 1, 0, \ldots, 0)$ is a standard basis vector, so that $\|x_i - x_j\|_1 = t$ for any $i \neq j$. Since the matrix of values $(f(\|x_i - x_j\|_1)_{i,j \in [N]}$ must be positive semidefinite, the sum of all its entries must be positive, hence:

$$N(f(0) + (N-1)f(t)) \geq 0.$$

The above equation implies the following:

$$\frac{f(0)}{N-1} + f(t) \geq 0$$

for all integer $N \geq 0$ and real $t \geq 0$.

Since $N$ can be arbitrarily large, therefore we conclude $f(t) \geq 0$ as claimed.

$\square$

### G.2 Positive Definite Manhattan Kernels are Completely Monotone

The goal of this section is to prove Theorem G.2.

**Theorem G.2** (Formal statement of Theorem 3.4). *$f : \mathbb{R}_{\geq 0} \to \mathbb{R}$ is a positive definite Manhattan kernel (Definition 3.1) if and only if $f(x)$ is completely monotone (Definition 3.2).*

*Proof.* First, we prove that if $f$ is a positive definite Manhattan kernel, then $f$ must be completely monotone. The converse direction is previously known, and is a consequence of Lemma A.5 and Theorem 3 of [SOW01][11].

Suppose that $f$ is a positive definite Manhattan kernel (Definition 3.1). Cauchy-Schwarz easily implies that $f(t) \leq f(0)$ for all $t$, so $f$ is bounded.. Now, if $x_1, \ldots, x_n$ correspond to $y_1, \ldots, y_n$ then

$$f(\|x_i - x_j\|_1) = \langle y_i, y_j \rangle$$
$$= f(0) - \frac{1}{2}\|y_i - y_j\|_2^2.$$

Therefore $2(f(0) - f(t))$ (equivalently, $f(0) - f(t)$) sends Manhattan distances to squared Euclidean distances. Therefore $f(0) - f(t)$ is Bernstein (Definition 4.3), by Theorem 4.4. Combining with Lemma G.1 we conclude that $f$ must be completely monotone (Definition 3.2).

$\square$

# H    Positive Definite Euclidean Kernels

The goal of this section is to prove the foundational classification of positive definite Euclidean kernels [Smo96, SOW01, Sch42]. We focus on proving the 'hard' direction, that a function $f$ is a positive definite Euclidean kernel only if $f(\sqrt{x})$ is completely monotone. Such a proof is straightforward from Theorem G.2, as seen below. For the other simpler direction, see the simple proof in Proposition 11 of [SSB+97].

**Theorem H.1.** *[Sch42] $f : \mathbb{R}_{\geq 0} \to \mathbb{R}$ is a positive definite Euclidean kernel (Definition 3.1) only if $f(\sqrt{x})$ is a completely monotone function (Definition 3.2).*

*Proof.* Theorem G.2 combined with the fact that Manhattan distances isometrically embed into squared Euclidean distances (see [DL09]) prove that $f(\sqrt{x})$ must be completely monotone if it is a positive definite Euclidean kernel. $\square$

# I    Eigenvalue of Kernels from the Hyperrectangle

In this section, we prove Lemma I.2, a variation of Lemma B.1. This lemma uses representation theoretic ideas to compute the eigenvalues of matrices arising from the real hyperrectangle. We use this modified formulation in our proofs of Theorems 4.4 and 3.4. We introduce Lemma I.3, which expresses these same eigenvalues in terms of integrals, a lemma of independent interest.

## I.1    Matrices with Reflectional Symmetries have Hadamard Eigenvectors

**Lemma I.1.** *Let $g : (\mathbb{R}^d \times \mathbb{R}^d) \to \mathbb{R}$ such that $g(x, y)$ is invariant under axis reflection. Consider a $d$-dimensional hyperrectangle with corners $x_1, \ldots x_{2^d}$. Let $D$ be a $2^d$ by $2^d$ matrix such that $D_{ij} = g(x_i, x_j)$. Then there is an eigendecomposition of $D$ into $H_d \Sigma H_d$ where $\Sigma$ is a diagonal matrix.*

*Proof.* This lemma can be proven directly via computation. However, it is more instructive to view this through the representation theoretic lens. We note that $D$ has the property that for any permutation matrix $\sigma$ corresponding to a reflection about one of the hyperrectangle's axes, we have $\sigma D = D\sigma$. Schur's lemma from representation theory (see Lemma A.10) states that $D$ and all $\sigma$ in the reflectional symmetry group of the hyperrectangle have a common set of eigenvectors. It is straightforward to verify that the only common set of eigenvectors for all $\sigma$ is the columns of the Hadamard matrix, and thus $D$ must have the columns of $H_d$ as its eigenvectors. $\square$

We note that variants of this lemma are used to prove Delsarte's linear programming bound in error correcting codes [Del73, O'D14].

## I.2    Eigenvalues of Kernels from the Hyperrectangle, Restated

**Lemma I.2** (Eigenvalue of Manhantan Kernels, formal version of Lemma B.1)**.** *Consider a $d$-dimensional hyperrectangle (Definition C.3) parameterized by $a_1, \ldots a_d > 0$. Enumerate the vertices in lexicographical ordering as $p_1, \ldots p_{2^d}$.*

*For any $f : \mathbb{R} \to \mathbb{R}$, let $D$ be the $2^d$ by $2^d$ matrix given by $D_{i,j} = f(\|p_i - p_j\|_1)$. Then:*

1. $\Sigma := H_d D H_d$ *is a diagonal matrix whose entries are the eigenvalues of $D$ multiplied by $2^d$, and $D = 4^{-d} \cdot H_d \Sigma H_d$.*

2. *Let $\chi : [d] \to \{0, 1\}$. Let $k$ equal the integer corresponding to transforming $\chi$ (written as a $d$ dimensional binary vector) into an integer via binary conversion. For each $\chi$, there is an eigenvector of $D$ equal to the $k$-th column of Hadamard matrix $H_d$, and its associated eigenvalue is:*

$$\sum_{T \subseteq [d]} (-1)^{\sum_{t \in T} \chi(t)} f\left(\sum_{t \in T} a_t\right). \tag{9}$$

The second part of this theorem on its surface differs from that in Lemma B.1, but the statements are in fact identical via straightforward computation.

*Proof.* By Lemma I.1, we know that the Hadamard matrix columns are eigenvectors of the matrix $D$. The result follows by direct computation. $\square$

We now give an alternate formulation of the eigenvalues in Lemma I.2. This lemma is of independent interest.

**Lemma I.3.** *Given a box with side lengths $a_1, \ldots a_d$, each eigenvalue corresponds to a function $\chi : [d] \to \{0, 1\}$. Let $Q = \{q_1, \ldots q_k\}$ be the full set of values on which $\chi$ is 1. Then the Eigenvalues in Eq. (9) equal:*

$$\sum_{T \subseteq [d] \backslash Q} \int_{\sum_{t \in T} a_t}^{a_{q_1} + \sum_{t \in T} a_t} \cdots \int_{\sum_{t \in T} a_t}^{a_{q_k} + \sum_{t \in T} a_k} (-1)^k \frac{\mathrm{d}^k f}{\mathrm{d}x^k} \left(\sum_{q \in Q} s_q\right) \mathrm{d}s_1 \ldots \mathrm{d}s_k.$$

*Proof.* The proof follows directly from Lemma I.2 combined with the fundamental theorem of calculus. $\square$

# J   Converse to Stable Rank

Recall Definition 2.7, for a matrix $A$, we use $\mathrm{srank}(A)$ to denote the stable rank of $A$. For a matrix $A$, we use $\|A\|_F$ to denote its Frobenius norm. We use $\|A\|$ to denote its spectral norm.

## J.1   Lipschitz functions preserve stable rank

**Definition J.1.** *We say function $f$ is $(L, \ell)$-Lipshitz on on entries of matrix $A$, if for any $x \geq 0$ in entries of $A$ such that*

$$\frac{1}{\sqrt{\ell}} x \leq f(x) \leq \sqrt{L} x$$

**Theorem J.2.** *We define $B \in \mathbb{R}_{\geq 0}^{n \times n}$ as follows $B_{i,j} = f(A_{i,j})$ for all $i \in [n]$ and for all $j \in [n]$.*

*If function $f$ is $(L, \ell)$-Lipschitz on $A \in \mathbb{R}_{\geq 0}^{n \times n}$, then we have the following*

- **Part 1.** $1/\sqrt{l} \cdot \|A\|_F \leq \|B\|_F \leq \sqrt{L} \cdot \|A\|_F$.

- **Part 2.** $1/\sqrt{l} \cdot \|A\| \leq \|B\| \leq \sqrt{L} \cdot \|A\|$.

- **Part 3.** $L^{-1} \cdot \ell^{-1} \cdot \mathrm{srank}(A) \leq \mathrm{srank}(B) \leq L \cdot \ell \cdot \mathrm{srank}(A)$.

*Proof.* **Proof of Part 1.**

We can upper bound $\|B\|_F^2$ as follows

$$\|B\|_F^2 = \sum_{i=1}^n \sum_{j=1}^n f(A_{i,j})^2$$
$$\leq L \cdot \sum_{i=1}^n \sum_{j=1}^n A_{i,j}^2$$
$$= L \cdot \|A\|_F^2 \tag{10}$$

where the first step follows from definition of $\|\cdot\|_F$, the second step follows from $f(A_{i,j})^2 \leq L \cdot A_{i,j}^2$, and the last step follows from definition of $\|\cdot\|_F$ norm.

We can lower bound $\|B\|_F^2$ as follows

$$\|B\|_F^2 = \sum_{i=1}^n \sum_{j=1}^n f(A_{i,j})^2$$
$$\geq l^{-1} \cdot \sum_{i=1}^n \sum_{j=1}^n A_{i,j}^2$$
$$= l^{-1} \cdot \|A\|_F^2 \tag{11}$$

where first step follows from definition of $\|\cdot\|_F$ norm, the second step follows from $f(A_{i,j})^2 \geq l^{-1} \cdot A_{i,j}^2$, and the last step follows from definition of $\|\cdot\|_F$ norm.

**Proof of Part 2.**

We can upper bound on $\|B\| = \sigma(B)$ as follows

$$\sigma(B)^2 = \max_{\|v\|_2=1} (v^\top B v)^2$$
$$= \max_{\|v\|_2=1} (\sum_{i=1}^n \sum_{j=1}^n v_i v_j f(A_{i,j}))^2$$
$$\leq \max_{\|v\|_2=1} (\sum_{i=1}^n \sum_{j=1}^n v_i v_j A_{i,j})^2 \cdot L$$
$$= \sigma(A)^2 \cdot L \tag{12}$$

where the first step follows definition of spectral norm, the third step follows from $f(A_{i,j})^2 \leq A_{i,j}^2 \cdot L$.

We can lower bound on $\|B\| = \sigma(B)$ as follows

$$\sigma(B)^2 = \max_{\|v\|_2=1} (v^\top B v)^2$$
$$= \max_{\|v\|_2=1} (\sum_{i,j} v_i v_j f(A_{i,j}))^2$$
$$\geq \max_{\|v\|_2=1} (\sum_{i,j} v_i v_j A_{i,j})^2 / \ell$$
$$= \sigma(A)^2 / \ell \tag{13}$$

where the third step follows from $f(A_{i,j})^2 \geq A_{i,j}^2 / \ell$.

**Proof of Part 3.**

We can upper bound $\text{srank}(B)$ as follows

$$
\begin{aligned}
\text{srank}(B) &= \frac{\|B\|_F^2}{\sigma(B)^2} \\
&\leq \frac{L \cdot \|A\|_F^2}{\sigma(B)^2} \\
&\leq \frac{L \cdot \|A\|_F^2}{\sigma(A)^2/\ell} \\
&= L \cdot \ell \cdot \text{srank}(A)
\end{aligned}
$$

where the first step follows from definition of stable rank, the second step follows from Eq. (10), the third step follows from Eq. (13), and the last step follows from stable rank.

We can lower bound $\text{srank}(B)$ as follows

$$
\begin{aligned}
\text{srank}(B) &= \frac{\|B\|_F^2}{\sigma(B)^2} \\
&\geq \frac{\|A\|_F^2/\ell}{\sigma(B)^2} \\
&\geq \frac{\|A\|_F^2/\ell}{\sigma(A)^2 L} \\
&= L^{-1} \cdot \ell^{-1} \cdot \text{srank}(A)
\end{aligned}
$$

where the first step follows from definition of stable rank, the second step follows from Eq. (11), the third step follows from Eq. (12), and the last step follows from definition of stable rank. $\qquad\square$

## J.2 Fast-Growing functions do not preserve stable rank

We start with presenting a tool for symmetric matrix.

**Lemma J.3.** *Consider a symmetric matrix $M$ with non-negative entries, with the all ones vector as an eigenvector. The eigenvalue of this vector is the largest eigenvalue of $M$*

*Proof.* This follows from the Perron Froebenius formula [Per07, Fro12] on non-negative matrices. $\qquad\square$

For a matrix $M$, we use $M_{i,j}$ to denote the entry at $i$-th row and $j$-th column in the matrix.

**Lemma J.4.** *Let $M$ be a $n$ by $n$ matrix with non-negative entries, with an eigenvector that is the alll ones vector. Suppose that there exists permutation $\sigma : [n] \to [n]$ such that $M_{i,\sigma(i)}$ is the unique largest element in row $i$ for all $i \in [n]$. Suppose that $k := M_{i,\sigma(i)}$ which is independent of $i$, and all other entries are less than $s$ where $s < k$.*

*Then, we have*

$$
\text{srank}(M) \geq \frac{nk^2}{(k+ns)^2}
$$

*Proof.* By Lemma J.3, the eigenvalue of $M$ corresponding to the all ones vector, is the largest eigenvalue of $M$. Recall that the stable rank of $M$ is defined as the Froebenius norm squared, divided by the spectral norm squared.

The largest eigenvalue of $M$ is $\sum_i M_{i1}$ which is bounded above by $k + sn$. Meanwhile, the squared Froebenius norm of $M$ is bounded above by $nk^2$, which is the sum of squares of the diagonal elements. This completes the proof. Dividing our two bounds gives our lemma. $\qquad\square$

Now, consider the distance matrix $M$ arising from $f$ applied entry-wise to the matrix arising from the hypercube with side lengths $\beta$ (this is the hyperrectangle where all side lengths are the same). We

note that the hypercube matrix has rank $\log n + 1$, and thus its stable rank is also bounded by this quantity.

In this case, we note that $M$ has non-negative entries, has the all ones vector as an eigenvector, and has the property that there exists a permutation $\sigma$ such that $M_{i,\sigma(i)}$ is $\beta \log n$. Meanwhile, all the other entries are less than $\beta \log n - \beta$.

We note that if $k/s > n^{0.5}$, then the stable rank of $M$ is $\Omega(n)$.

The rest of the proof is devoted to understanding when $k/s > n^{0.5}$.

This is equivalent to:

$$\frac{f(\beta \log n)}{f(\beta \log n - \beta)} \geq n^{0.5} \tag{14}$$

Since $\beta$ can be set to anything, we define $\gamma := \beta \log n$, and thus Eq. (14) is equivalent to

$$f(\gamma(1 + \frac{1}{\log n})) \geq n^{0.5} f(\gamma). \tag{15}$$

**Theorem J.5** (Superpolynomials don't preserve stable rank). $f(x) = x^{\log^c x} + o(x^{\log^c x})$ *does not preserve stable rank for any $c > 0$.*

*Proof.* It is sufficient to show that

$$\frac{f(\gamma(1 + \epsilon))}{f(\gamma)} \tag{16}$$

is unbounded for fixed $\epsilon$ and variable $\gamma > 0$.

Substituting $f(x) = x^{\log^c x}$, we get

$$\frac{\gamma^{\log^c(\gamma + \gamma\varepsilon)}}{\gamma^{\log^c \gamma}} = 2^{(\log \gamma)\cdot(\log^c(\gamma + \gamma\varepsilon) - \log^c(\gamma))} \tag{17}$$

Next, we need to show that

$$\lim_{\gamma \to \infty} \text{Eq. (17)} = \infty.$$

We define function

$$F(x) = \log^c(x).$$

and compute

$$F'(x) = c \cdot \frac{1}{x} \log^{c-1}(x),$$

$$F''(x) = c \cdot (c-1)\frac{1}{x^2} \log^{c-2}(x) - c\frac{1}{x^2} \log^{c-1}(x)$$

$$= c\frac{1}{x^2}((c-1)\log^{c-2}(x) - \log^{c-1}(x))$$

Using mean-value forms of Taylor's theorem,

$$F(y) = F(x) + F'(x) \cdot (y - x) + \frac{1}{2}F''(z) \cdot (y - x)^2$$

where $z \in [x, y]$.

We can compute

$$F'(x) \cdot (y - x) = c\frac{1}{\gamma} \log^{c-1}(\gamma)(\epsilon\gamma)$$

$$= \epsilon c \log^{c-1}(\gamma)$$

We can compute

$$\frac{1}{2}F''(z) \cdot (y-x)^2 = \frac{1}{z^2}((c-1)\log^{c-2}(z) - \log^{c-1}(z)) \cdot (\epsilon\gamma)^2$$

$$= \frac{\epsilon^2}{(1+\alpha)^2}((c-1)\log^{c-2}(z) - \log^{c-1}(z))$$

where choosing $z = \alpha\gamma$ (where $\alpha \in [1, (1+\epsilon)]$).

Then it is obvious to see that $|\frac{1}{2}F''(z) \cdot (y-x)^2| \leq \frac{1}{10}F'(x) \cdot (y-x)$.

Finally, we can lower bound

$$F(\gamma(1+\epsilon)) - F(\gamma) \geq F'(\gamma) \cdot \epsilon\lambda - \frac{1}{2}|F''(z)| \cdot (\epsilon\gamma)^2$$

$$\geq \frac{1}{2}F'(\gamma) \cdot \epsilon\lambda$$

$$= \frac{1}{2}\epsilon c \log^{c-1}(\gamma)$$

where $z \in [\gamma, (1+\epsilon)\gamma]$

Thus, we have

$$\lim_{\gamma \to \infty} \text{Eq. (17)} = \lim_{\gamma \to \infty} 2^{(\log \gamma) \cdot (\log^c(\gamma+\gamma\varepsilon) - \log^c(\gamma))}$$

$$= \lim_{\gamma \to \infty} 2^{(\log \gamma) \cdot \frac{1}{2}\epsilon c \log^{c-1}(\gamma)}$$

$$= \infty.$$

Thus, we complete the proof.

$\square$

